# Diffusion-based Curriculum Reinforcement Learning

**Erdi Sayar[1], Giovanni Iacca[2], Ozgur S. Oguz[3], Alois Knoll[1]**
Technical University of Munich[1] University of Trento[2] Bilkent University[3]
erdi.sayar@tum.de, giovanni.iacca@unitn.it,
ozgur@cs.bilkent.edu.tr, knoll@in.tum.de

## Abstract

Curriculum Reinforcement Learning (CRL) is an approach to facilitate the learning process of agents by structuring tasks in a sequence of increasing complexity. Despite its potential, many existing CRL methods struggle to efficiently guide agents toward desired outcomes, particularly in the absence of domain knowledge. This paper introduces DiCuRL (Diffusion Curriculum Reinforcement Learning), a novel method that leverages conditional diffusion models to generate curriculum goals. To estimate how close an agent is to achieving its goal, our method uniquely incorporates a $Q$-function and a trainable reward function based on Adversarial Intrinsic Motivation within the diffusion model. Furthermore, it promotes exploration through the inherent noising and denoising mechanism present in the diffusion models and is environment-agnostic. This combination allows for the generation of challenging yet achievable goals, enabling agents to learn effectively without relying on domain knowledge. We demonstrate the effectiveness of DiCuRL in three different maze environments and two robotic manipulation tasks simulated in MuJoCo, where it outperforms or matches nine state-of-the-art CRL algorithms from the literature.

## 1 Introduction

Reinforcement learning (RL) is a computational method that allows an agent to discover optimal actions through trial and error by receiving rewards and adapting its strategy to maximize cumulative rewards. Deep RL, which integrates deep neural networks (NNs) with RL, is an effective way to solve large-dimensional decision-making problems, such as learning to play video games [1, 2], chess [3], Go [4], and robot manipulation tasks [5, 6, 7, 8]. One of the main advantages of deep RL is that it can tackle difficult search problems where the expected behaviors and rewards are often sparsely observed. The drawback, however, is that it typically needs to thoroughly explore the state space, which can be costly especially when the dimensionality of this space grows.

Some methods, such as reward shaping [9], can mitigate the burden of exploration, but they require domain knowledge and prior task inspection, which limits their applicability. Alternative strategies have been proposed to enhance the exploration efficiency in a domain-agnostic way, such as prioritizing replay sampling [8, 10, 11], or generating intermediate goals [12, 13, 14, 15, 16, 17, 18, 19]. This latter approach, known as *Curriculum Reinforcement Learning* (CRL), focuses on designing a suitable curriculum to guide the agent gradually toward the desired goal.

Various approaches have been proposed for the generation of curriculum goals. Some methods focus on interpolation between a source task distribution and a target task distribution [20, 21, 22, 17]. However, these methods often rely on assumptions that may not hold in complex RL environments, such as specific parameterization of distributions, hence ignoring the manifold structure in space. Other approaches adopt optimal transport [13, 23], but they are typically applied in less challenging exploration scenarios. Curriculum generation based on uncertainty awareness has also been explored, but such methods often struggle with identifying uncertain areas as the goal space expands [15, 24, 12].

Some research minimizes the distance between generated curriculum and desired outcome distributions using Euclidean distance, although this approach can be problematic in certain environments [19, 25]. Other methods incorporate graph-based planning, but require an explicit specification of obstacles [26, 27]. Lastly, approaches based on generative AI models have been proposed. For instance, [14] uses GANs to generate tasks of intermediate difficulty, but it relies on arbitrary thresholds. Alternatively, [28, 29, 30] apply diffusion models in offline RL settings [14, 28, 30].

Despite these advancements, existing CRL approaches still struggle with generating suitable intermediate goals, particularly in complex environments with significant exploration challenges. To overcome this challenge, in this paper, we propose DICURL (Diffusion Curriculum Reinforcement Learning). Our method leverages conditional diffusion models to dynamically generate curriculum goals, guiding agents towards desired goals while simultaneously considering the $Q$-function and a trainable reward function based on Adversarial Intrinsic Motivation (AIM) [31].

**Contributions** Unlike previous offline RL approaches [28, 29, 30] that train and use diffusion models for planning or policy generation relying on pre-existing data, DICURL facilitates *online learning*, enabling agents to learn effectively without requiring domain-specific knowledge. This is achieved by three key elements. ① The diffusion model captures the *distribution of visited states* and facilitates exploration through its inherent noising and denoising mechanism. ② As the $Q$-function predicts the cumulative reward starting from a state and a given goal while following a policy, we can determine *feasible* goals by maximizing the $Q$-function, ensuring that the generated goals are challenging yet achievable for the agent. ③ The AIM reward function estimates the agent's proximity to the desired goal and allows us to *progressively shift the curriculum* towards the desired goal.

We compare our proposed approach with nine state-of-the-art CRL baselines in three different maze environments and two robotic manipulation tasks simulated in MuJoCo [32]. Our results show that DICURL surpasses or performs on par with the state-of-the-art CRL algorithms.

## 2   Related Work

**Curriculum Reinforcement Learning** CRL [33] algorithms generally adjust the sequence of learning experiences to improve the agent's performance or accelerate training. These algorithms focus on formulating intermediate goals that progressively guide the agent toward the desired goal, and have been successfully applied to various tasks, mainly in the field of robot manipulation [34, 35, 36, 37].

Hindsight Experience Replay (HER) [38] tackles the challenge of sparse reward RL tasks by employing hindsight goals, considering the achieved goals as pseudo-goals, and substituting them for the desired goal. However, HER struggles to solve tasks when the desired goals are far from the initial position. Hindsight Goal Generation (HGG) [19] addresses the inefficiency issue inherent in HER by generating hindsight goals through maximizing a value function and minimizing the Wasserstein distance between the achieved goal and the desired goal distribution.

CURROT [23] and GRADIENT [13] both employ optimal transport for the generation of intermediate goals. CURROT formulates CRL as a constrained optimization problem and uses the Wasserstein distance to measure the distance between distributions. Conversely, GRADIENT introduces task-dependent contextual distance metrics and can manage non-parametric distributions in both continuous and discrete context settings; moreover, it directly interprets the interpolation as the geodesic from the source to the target distribution.

GOAL-GAN [14] generates intermediate goals using a Generative Adversarial Network (GAN) [39], without considering the target distribution. A goal generator is used to propose goal regions, and a goal discriminator is trained to evaluate if a goal is at the right level of difficulty for the current policy. The specification of goal regions is done using an indicator reward function, and policies are conditioned on the goal as well as the state, similarly to a universal value function approximator [40].

PLR [15] uses selective sampling to prioritize instances with higher estimated learning potential for future revisits during training. Learning potential is estimated using TD-Errors, resulting in the creation of a more challenging curriculum. VSD [16] estimates the epistemic uncertainty of the value function and selects goals based on this uncertainty measure. The value function confidently assigns high values to easily achievable goals and low values to overly challenging ones. ACL [18] maximizes the learning progress by considering two main measures: the rate of improvement in prediction accuracy and the rate of increase in network complexity. This signal acts as an indicator of

the current rate of improvement of the learner. The ALP-GMM [17] fits Gaussian Mixture Models (GMM) using an Absolute Learning Progress (ALP) score, which is defined as the absolute difference in rewards between the current episode and the previous episodes. The teacher generates curriculum goals by sampling environments to maximize the student's ALP, which is modeled by the GMM.

Finally, OUTPACE [12] employs a trainable intrinsic reward mechanism, known as Adversarial Intrinsic Motivation (AIM) [31] (the same used in our method) which is designed to minimize the Wasserstein distance between the state visitation distribution and the goal distribution. This function increases along the optimal goal-reaching trajectory. For curriculum goal generation, OUTPACE uses Conditional Normalized Maximum Likelihood (CNML), to classify state success labels based on their association with visited states, out-of-distribution samples, or the desired goal distribution. The method also prioritizes uncertain and temporally distant goals using meta-learning-based uncertainty quantification [41] and Wasserstein-distance-based temporal distance approximation.

**Diffusion Models for Reinforcement Learning** UniPi [42] leverages diffusion models to generate a video as a planner, conditioned on an initial image frame and a text description of a current goal. Subsequently, a task-specific policy is employed to infer action sequences from the generated video using an inverse dynamic model. AVDC [43] constructs a video-based robot policy by synthesizing a video that renders the desired task execution and directly regresses actions from the synthesized video without requiring any action labels or inverse dynamic model. It takes RGBD observations and a textual goal description as inputs, synthesizes a video of the imagined task execution using a diffusion model, and estimates the optical flow between adjacent frames in the video. Then, using the optical flow and depth information, it computes robot commands.

Diffusion Policy [44] uses a diffusion model to learn a policy through a conditional denoising diffusion process. BESO [45] adopts an imitation learning approach that learns a goal-specified policy without any rewards from an offline dataset. DBC [46] uses a diffusion model to learn state-action pairs sampled from an expert demonstration dataset and increases generalization using the joint probability of the state-action pairs. Finally, Diffusion BC [47] uses a diffusion model to imitate human behavior and capture the full distribution of observed actions on robot control tasks and 3D gaming environments.

**Limitations of current works and distinctive aspects of DICURL** The aforementioned studies typically require offline data for training. Both [42] and [43] employ diffusion models to synthesize videos for rendering the desired task execution, and actions are then inferred from such videos. Studies such as [44, 45, 46, 47] also focus on learning policies from offline datasets. Despite these efforts, reliance on inadequate demonstration data can lead to suboptimal performance [44]. Distinct from these approaches, our method instead does not rely on prior expert data or any pre-collected datasets. As an off-policy RL method, DICURL collects in fact data through interaction with the environment.

## 3 Background

We now introduce the background concepts on multi-goal RL, Soft Actor-Critic (SAC), Wasserstein distance, Adversarial Intrinsic Motivation (AIM), and diffusion models.

### 3.1 Multi-Goal Reinforcement Learning

In the context of multi-goal RL, we can formulate the RL problem as a goal-oriented Markov decision process (MDP) characterized by continuous state and action spaces. This MDP is defined by the tuple $\langle \mathcal{S}, \mathcal{A}, \mathcal{G}, \mathcal{T}, p, \gamma_r \rangle$. Here, $\mathcal{S}$ represents the state space, $\mathcal{A}$ denotes the action space, and $\mathcal{G}$ is the goal space. The transition dynamics, $\mathcal{T}(s'|s, a)$, describes the probability of transitioning to the next state $s'$ given the current state $s$ and action $a$. The joint probability distribution over the initial states and the desired goal distribution is represented by $p(s_0, g)$, and $\gamma_r \in [0, 1]$ is a discount factor.

We utilize the AIM reward function $r_\varphi$, as outlined in Section 3.3. The objective is to identify the optimal policy $\pi$ that maximizes the expected cumulative reward approximated by the value function $Q^\pi(s, g, a)$. This function can be expanded to the Universal Value Function (UVF), a goal-conditioned value function incorporating the goal into value estimation. The UVF, defined as $V^\pi(s, g)$, where $s$ is the current state, $g$ is the goal, and $\pi$ is the policy, estimates the expected return

from state $s$ under policy $\pi$, given goal $g$. The UVF has been shown to successfully generalize to unseen goals, which makes it a viable approach for multi-goal RL [40].

## 3.2 Soft Actor-Critic

Soft Actor-Critic (SAC) [48] is an off-policy RL algorithm that learns a policy maximizing the sum of the expected discounted cumulative reward and the entropy of the policy, namely: $J(\pi) = \sum_{t=0}^{T} \mathbb{E}_{s_t \sim \mathcal{B}} \ r + \alpha_e \mathcal{H}(\pi(\cdot \mid s_t, g))$. The policy $\pi_\psi(a \mid s) : \mathcal{S} \mapsto \mathcal{P}(\mathcal{A})$ defines a map from state to distributions over actions, parameterized by $\psi$. A state-action value function is defined as $Q_\phi(s, g, a) : \mathcal{S} \times \mathcal{G} \times \mathcal{A} \to \mathbb{R}$, parameterized by $\phi$. The policy parameters can be learned by directly minimizing the expected KL divergence and the $Q$-value is trained by minimizing the soft Bellman residual, which are defined as the following loss functions:

$$\mathcal{L}_\pi = \mathbb{E}_{(s,g) \sim \mathcal{B}} \left[ \mathbb{E}_{a \sim \pi_\psi} \left[ \alpha_e \log \left( \pi_\psi(a \mid s, g) - Q_\phi(s, g, a) \right) \right] \right] \tag{1}$$

$$\mathcal{L}_Q = \mathbb{E}_{(s,a,r,s',g) \sim \mathcal{B}} \left[ Q_\phi(s, g, a) - (r + \gamma_r \mathbb{E}_{s' \sim \mathcal{B}} \left[ V_\phi(s', g) \right])^2 \right] \tag{2}$$

where $V_\phi(s, g) = \mathbb{E}_{a \sim \pi} \left[ Q_\phi(s, g, a) - \log \pi_\psi(a \mid s, g) \right]$, $\mathcal{B}$ is the replay buffer and $\alpha_e$ is the temperature parameter that controls the stochasticity of the optimal policy.

## 3.3 Wasserstein Distance and Adversarial Intrinsic Motivation Reward Function

The Wasserstein distance offers a way to quantify the amount of work required to transport one distribution to another distribution. The Wasserstein-p distance $W_p$ between two distributions $\mu$ and $\nu$ on a metric space $(\mathcal{X}, d)$, where $\mathcal{X}$ is a set and $d$ denotes a metric on $\mathcal{X}$, is defined as follows [49]:

$$W_p(\mu, \nu) := \inf_{\gamma \in \Pi(\mu,\nu)} \left( \int_{\mathcal{X}} d(x, y)^p d\gamma(x, y) \right)^{1/p} = \inf_{\gamma \in \Pi(\mu,\nu)} \mathbb{E}_{(X,Y) \sim \gamma} \left[ d(X, Y)^p \right]^{1/p} \tag{3}$$

where $\Pi(\mu, \nu)$ denotes the set of all possible joint distributions $\gamma(x, y)$ whose marginals are $\mu$ and $\nu$. Intuitively, $\gamma(x, y)$ tells what is the least amount of work, as measured by $d$, that needs to be done to convert the distribution $\mu$ into the distribution $\nu$ [50]. A timestep quasi-metric $d^\pi(s, g)$ can be used as a distance metric. It estimates the work needed to transport one distribution to another, representing the number of transition steps required to reach the goal state $g \in \mathcal{G}$ for the first time when following the policy $\pi$.

As proposed by Durugkar et al. [31], the AIM reward function can be learned by minimizing the Wasserstein distance between the state visitation distribution $\rho_\pi$ and the desired goal distribution $\mathcal{G}$. Through the minimization of the Wasserstein distance $W_1(\rho_\pi, \mathcal{G})$ (for $p = 1$, $W_1$ is known as the Kantorovich–Rubinstein distance), a reward function can be formulated to estimate the work required to transport the state visitation distribution $\rho_\pi$ to the desired goal distribution $\mathcal{G}$ and is expressed as follows: $W_1(\rho_\pi, \mathcal{G}) = \sup_{\|f\|_L \leq 1} \left[ \mathbb{E}_{g \sim \mathcal{G}} \left[ f(g) \right] - \mathbb{E}_{s \sim \rho_\pi} \left[ f(s) \right] \right]$. If the state visitation distribution $\rho_\pi(s)$ comprises states that optimally progress towards the goal $g$, the potential function $f(s)$ increases along the trajectory, reaching its maximum value at $f(g)$. The reward function, which can be approximated using a neural network, denoted as $r_\varphi^\pi$, increases as the states approach the desired goal $g \in \mathcal{G}$. $r_\varphi^\pi$ can be trained using the data collected by the policy $\pi$. Leveraging the estimation of the Wasserstein distance $W_1(\rho_\pi, \mathcal{G})$, the loss function for training the parameterized reward function $r_\varphi^\pi$ is defined as follows:

$$\mathcal{L}_\varphi = \mathbb{E}_{(s,g) \sim \mathcal{B}} \left[ f_\varphi^\pi(s) - f_\varphi^\pi(g) \right] + \lambda \cdot \mathbb{E}_{(s,s',g) \sim \mathcal{B}} \left[ \max \left( \left| f_\varphi^\pi(s) - f_\varphi^\pi(s') \right| - 1, 0 \right)^2 \right] \tag{4}$$

where the second component of the sum is a penalty term and the coefficient $\lambda$ is necessary to ensure smoothness [31]. The reward can be calculated as $r_\varphi(s, g) = f_\varphi^\pi(s) - f_\varphi^\pi(g)$, which is the negative of the Wasserstein distance $-W_1(\rho_\pi, \mathcal{G})$.

## 3.4 Diffusion Models

Diffusion models [51] express a probability distribution $p(x_0)$ through latent variables in the form $p_\theta(\boldsymbol{x}_0) := \int p_\theta(\boldsymbol{x}_{0:N}) \, d\boldsymbol{x}_{1:N}$, where $\boldsymbol{x}_1, \ldots, \boldsymbol{x}_N$ are latent variables of the same dimensionality as the data $\boldsymbol{x}_0 \sim p(\boldsymbol{x}_0)$. They are characterized by a forward and a reverse diffusion process.

The forward diffusion process approximates the posterior $q\left(\boldsymbol{x}_{1:N} \mid \boldsymbol{x}_0\right)$ using a Markov chain that perturbs the input data by gradually adding Gaussian noise $\boldsymbol{x}_0 \sim q\left(\boldsymbol{x}_0\right)$ in $N$ steps with a predefined variance schedule $\beta_1, \ldots, \beta_N$. This process is defined as:

$$q\left(\boldsymbol{x}_{1:N} \mid \boldsymbol{x}_0\right) := \prod_{k=1}^{N} q\left(\boldsymbol{x}_k \mid \boldsymbol{x}_{k-1}\right) \tag{5}$$

$$q\left(\boldsymbol{x}_k \mid \boldsymbol{x}_{k-1}\right) := \mathcal{N}\left(\boldsymbol{x}_k; \sqrt{1 - \beta_k} \boldsymbol{x}_{k-1}, \beta_k \boldsymbol{I}\right)$$

The reverse diffusion process aims to recover the original input data from the noisy (diffused) data. It learns to progressively reverse the diffusion process step by step and approximates the joint distribution $p_\theta\left(\boldsymbol{x}_{0:N}\right)$. This process is defined as:

$$p_\theta\left(\boldsymbol{x}_{0:N}\right) := p\left(\boldsymbol{x}_N\right) \prod_{k=1}^{N} p_\theta\left(\boldsymbol{x}_{k-1} \mid \boldsymbol{x}_k\right) \tag{6}$$

$$p_\theta\left(\boldsymbol{x}_{k-1} \mid \boldsymbol{x}_k\right) := \mathcal{N}\left(\mathbf{x}_{k-1}; \boldsymbol{\mu}_\theta\left(\mathbf{x}_k, k\right), \boldsymbol{\Sigma}_\theta\left(\mathbf{x}_k, k\right)\right)$$

where $p\left(\boldsymbol{x}_N\right) = \mathcal{N}\left(\boldsymbol{x}_N; \mathbf{0}, \boldsymbol{I}\right)$. The optimization of the reverse diffusion process is achieved by maximizing the evidence lower bound (ELBO) $\mathbb{E}_q\left[\ln \frac{p_\theta(\boldsymbol{x}_{0:N})}{q(\boldsymbol{x}_{1:N}|\boldsymbol{x}_0)}\right]$ [52]. Once trained, sampling data from Gaussian noise $\boldsymbol{x}_N \sim p\left(\boldsymbol{x}_N\right)$ and running through the reverse diffusion process from $k = N$ to $k = 0$ yields an approximation of the original data distribution.

## 4 Methodology

As discussed earlier, in multi-goal RL, the desired goal is sampled from a desired goal distribution in each episode, and the agent aims to achieve multiple goals. By integrating a curriculum design into multi-goal RL, we can reformulate a task in such a way that it starts with easier goals and progressively increases in difficulty. Our curriculum diffusion-model-based goal-generation method works as follows. Given the presence of two different types of timesteps for the diffusion process and the RL task, we denote the diffusion timesteps using subscripts $k \in \{1, \ldots, N\}$ and the RL trajectory timesteps using subscripts $t \in \{1, \ldots, T\}$. Our curriculum goal generation takes the state $s$ as an input. It then outputs a curriculum goal set $\mathcal{G}_c$, which is obtained through the reverse diffusion process of a conditional diffusion model as follows:

$$\mathcal{G}_c = p_\theta\left(\mathbf{g}_{0:N} \mid s\right) = \mathcal{N}\left(\mathbf{g}_N; \mathbf{0}, \boldsymbol{I}\right) \prod_{k=1}^{N} p_\theta\left(\mathbf{g}_{k-1} \mid \mathbf{g}_k, \boldsymbol{s}\right) \tag{7}$$

where $\mathbf{g}_0$ is the end sample of the reverse diffusion process used as a curriculum goal. Commonly, $p_\theta\left(\mathbf{g}_{k-1} \mid \mathbf{g}_k, \boldsymbol{s}\right)$ is a conditional distribution parametrized by $\theta$ and is chosen to model a multivariate Gaussian distribution $\mathcal{N}\left(\mathbf{g}_{k-1}; \boldsymbol{\mu}_\theta\left(\mathbf{g}_k, \boldsymbol{s}, k\right), \boldsymbol{\Sigma}_\theta\left(\mathbf{g}_k, \boldsymbol{s}, k\right)\right)$. Following [51], rather than learning the variances of the forward diffusion process, we assign a fixed covariance matrix $\boldsymbol{\Sigma}_\theta\left(\mathbf{g}_i, \boldsymbol{s}, i\right) = \beta_i \boldsymbol{I}$ and a mean defined as:

$$\boldsymbol{\mu}_\theta\left(\mathbf{g}_k, \boldsymbol{s}, k\right) = \frac{1}{\sqrt{\alpha_k}}\left(\mathbf{g}_k - \frac{\beta_k}{\sqrt{1 - \bar{\alpha}_k}} \boldsymbol{\epsilon}_\theta\left(\mathbf{g}_k, \boldsymbol{s}, k\right)\right) . \tag{8}$$

Initially, we sample a Gaussian noise $\mathbf{g}_N \sim \mathcal{N}(\mathbf{0}, \boldsymbol{I})$. We then apply the reverse diffusion process parameterized by $\theta$ starting from the last step $N$ and proceeding backward to step 1:

$$\mathbf{g}_{k-1} \mid \mathbf{g}_k = \frac{\mathbf{g}_k}{\sqrt{\alpha_k}} - \frac{\beta_k}{\sqrt{\alpha_k\left(1 - \bar{\alpha}_k\right)}} \boldsymbol{\epsilon}_\theta\left(\mathbf{g}_k, \boldsymbol{s}, k\right) + \sqrt{\beta_k} \boldsymbol{\epsilon} \tag{9}$$

$$\boldsymbol{\epsilon} \sim \mathcal{N}(\mathbf{0}, \boldsymbol{I}), \text{ for } k = N, \ldots, 1 .$$

For the case $k = 1$ (last term of the reverse diffusion process), we set $\boldsymbol{\epsilon}$ to $\mathbf{0}$, to enhance the sampling quality by ignoring $\sqrt{\beta_k}$ and edge effects. In fact, as empirically demonstrated by [51], training the diffusion model yields better results when utilizing a simplified loss function that excludes the weighting term. Thus, we adopt the following simplified loss function to train the conditional $\epsilon_\theta$-model, where $\epsilon_\theta$ is a function approximator designed to predict $\epsilon$[1].

$$\mathcal{L}_d(\theta) = \mathbb{E}_{k \sim \mathcal{U}, \boldsymbol{\epsilon} \sim \mathcal{N}(\mathbf{0}, \boldsymbol{I}), (\boldsymbol{s}, \mathbf{g_0}) \sim \mathcal{B}} \left[ \left\| \boldsymbol{\epsilon} - \boldsymbol{\epsilon}_\theta \left( \sqrt{\bar{\alpha}_k} \mathbf{g_0} + \sqrt{1 - \bar{\alpha}_k} \boldsymbol{\epsilon}, \boldsymbol{s}, k \right) \right\|^2 \right], \qquad (10)$$

where $\mathcal{U}$ is a uniform distribution over the discrete set $\{1, \ldots, N\}$ and $\mathcal{B}$ denotes the replay buffer collected by policy $\pi$.

To sample feasible curriculum goals for the agent, we integrate the $Q$-function and the AIM reward function into the loss. The total loss function for training the diffusion model is defined as follows:

$$\mathcal{L} = \xi_d \cdot \mathcal{L}_d(\theta) - \mathbb{E}_{s,a \sim \mathcal{B}, \mathbf{g_0} \sim \mathcal{G}_c, g_d \sim \mathcal{G}} [\xi_q \cdot Q_\phi(s, \mathbf{g_0}, \pi(s, \mathbf{g_0})) + \xi_r \cdot r_\varphi(\mathbf{g_0}, g_d)] \qquad (11)$$

The rationale is that, while minimizing the loss $\mathcal{L}_d$ allows us to accurately capture the state distribution, we also aim to simultaneously maximize the expected value of $Q$ and the AIM reward function $r_\varphi$, using the weights $\xi_d$, $\xi_q$, and $\xi_r$ to adjust the relative importance of the three components of the loss function. More in detail, the $Q$-function predicts the cumulative reward starting from a state and following the policy, while the AIM reward function estimates how close an agent is to achieving its goal. By maximizing $Q$ and the AIM reward, we can generate curriculum goals that are neither overly simplistic nor excessively challenging, progressing towards the desired goal.

In Eq. (11), $\mathbf{g_0}$ is obtained by sampling experiences from the replay buffer using Eq. (9) through a reverse diffusion process parameterized by $\theta$. Therefore, taking the gradient of $Q$ and the AIM reward $r_\varphi$ involves back-propagating through the entire diffusion process. After we obtain the set of curriculum goals $\mathcal{G}_c$ from the diffusion model, we use a bipartite graph $G(\{V_x, V_y\}, E)$ with edge costs $w$, composed of vertices $V_x$ (i.e., the curriculum goal candidates derived from the diffusion model) and vertices $V_y$ (i.e., the desired goals), where $E$ represents the edge weights, and select the optimal curriculum goal $g_c$[2]. We employ the Minimum Cost Maximum Flow algorithm in order to solve the bipartite matching problem and identify the edges with the minimal cost $w$:

$$\max_{\hat{\mathcal{G}}^c : |\hat{\mathcal{G}}^c| = K} \sum_{g_{t=0,\ldots T}^0 \in \hat{\mathcal{G}}^d, g_+^i \in \hat{\mathcal{G}}^+} w \quad \text{where:} \quad w = \sqrt{\frac{(g_i - \bar{g})^2}{N}} \; . \qquad (12)$$

The specifics of the generation of intermediate goals through diffusion models are explained in Algorithm 1, while the overall algorithm is outlined in Algorithm 2.

In Algorithm 1, during the training iterations, we sample Gaussian noise $\mathbf{g}_N \sim \mathcal{N}(\mathbf{0}, \boldsymbol{I})$ to denoise the data in the reverse diffusion process. Between lines 5 and 7, we iterate from time step $N$ to 1, where we perform the reverse diffusion process using Eq. (9) and subtract the predicted noise $\epsilon_\theta$ from $\mathbf{g}_N$ to denoise the noisy goal iteratively. We then uniformly sample a timestep $k$ from the range between 1 and $N$ (line 8) and sample a Gaussian noise $\epsilon \sim \mathcal{N}(\mathbf{0}, \boldsymbol{I})$ (line 9) in order to calculate the diffusion loss defined in Eq. (10). In line 10, we calculate the diffusion loss, the $Q$-value, and the AIM reward function using the generated goal $\mathbf{g_0}$ from the reverse diffusion process. Then in line 11, we calculate the total loss defined in Eq. (11) and update the diffusion model parameters $\theta$ using gradient descent. Finally, we return the generated goal $\mathbf{g_0}$.

In Algorithm 2 we begin by defining an off-policy algorithm denoted as $\mathbb{A}$. While any off-policy algorithm could be employed, we choose Soft Actor-Critic (SAC) to align with the baseline algorithms tested in our experiments. In line 5, we sample the initial state and provide the curriculum goal $g_c$ to the policy, along with the current state. The policy generates an action (line 7), and executes it in the environment (line 8). Subsequently, the next state and reward are obtained (line 9). Then, we provide the minibatch $b$ to the curriculum goal generator in line 11 to generate a curriculum goal set $\mathcal{G}_c$. Then we find the optimal curriculum goal $g_c$ using bipartite graph optimization (line 12). Furthermore, the loss functions defined in Eq. (1) and Eq. (2) are calculated, and the networks approximating $\pi$ and $Q$, as well as the AIM reward function $r_\varphi$ defined in Eq. (4), are updated. Between line 15 and 21, we run $n$ test rollouts and extract the achieved state of the agent using $\phi$ and calculate the success rate in reaching the desired goal within a threshold value $\kappa$.

**Algorithm 1** Diffusion Curriculum Goal Generator
---
1: **Input:** state $s$, no. of reverse diffusion timestep $N$, training step $M$
2: Obtain states $s$ from the minibatch $b$
3: **for** i $= 1, \ldots, M$ **do**                           ▷ Training iterations
4:      $\mathbf{g}_N \sim \mathcal{N}(\mathbf{0}, \boldsymbol{I})$
5:      **for** k $= N, \ldots, 1$ **do**                    ▷ Reverse diffusion process
6:          $\epsilon \sim \mathcal{N}(\mathbf{0}, \boldsymbol{I})$
7:          $g_{k-1} = \frac{1}{\sqrt{\alpha_k}} \left( g_k - \frac{\beta_k}{\sqrt{1-\bar{\alpha}_k}} \epsilon_\theta(g_k, k, s) \right) + \sqrt{\beta_k}\epsilon$      ▷ Using Eq. (9)
8:      $k \sim \text{Uniform}(\{1, \ldots, N\})$
9:      $\epsilon \sim \mathcal{N}(\mathbf{0}, \boldsymbol{I})$
10:      Calculate diffusion $\mathcal{L}_d(\theta), Q(s, \mathbf{g}_0, \pi(s)), r(g_d, \mathbf{g}_0)$ ▷ Calculate with the generated goal $\mathbf{g}_0$
11:      Calculate total loss $\mathcal{L} = \xi_d \mathcal{L}_d(\theta) - \xi_q Q(s, \mathbf{g}_0, \pi(s)) - \xi_r r(\mathbf{g}_0, g_d)$      ▷ Eq 11
12:      $\theta \leftarrow \theta - \eta \nabla_\theta \mathcal{L}$      ▷ Take gradient descent step and update the diffusion weights
     **return** $\mathbf{g}_0$
---

---
**Algorithm 2** RL Training and Evaluation
---
1: **Input:** no. of episodes $E$, timesteps $T$
2: Select an off-policy algorithm $\mathbb{A}$                              ▷ In our case, $\mathbb{A}$ is SAC
3: Initialize replay buffer $\mathcal{B} \leftarrow \emptyset$, $g_c \leftarrow \{g_d\}$ and networks $Q_\phi, \pi_\psi, r_\varphi$
4: **for** episode $= 0 \ldots E$ **do**
5:      Sample initial state $s_0$
6:      **for** $t = 0 \ldots T$ **do**
7:          $a_t = \pi(s_t, g_c)$
8:          Execute $a_t$, obtain next state $s_{t+1}$
9:          Store transition $(s_t, a_t, r_t, s_{t+1}, g_c)$ in $\mathcal{B}$
10:      Sample a minibatch $b$ from replay buffer $\mathcal{B}$
11:      $\mathcal{G}_c \leftarrow \texttt{DiffusionCurriculumGenerator}(b)$
12:      Find $g_c$ that maximizes $w$ in Eq. (12)
13:      Update $Q$ and $\pi$ with $b$ to minimize $\mathcal{L}_Q$ and $\mathcal{L}_\pi$ in Eq. (1) and in Eq. (2)
14:      Update the AIM reward function $r_\varphi$
15: $success \leftarrow 0$                                       ▷ Success rate
16: Sample a desired goal $g_d \sim \mathcal{G}$
17: **for** i $= 1 \ldots n_{testrollout}$ **do**
18:      $a_t = \pi(s_t, g_d)$
19:      Execute $a_t$, obtain next state $s_{t+1}$ and reward $r_t$
20:      **if** $|\phi(s_{t+1}) - g_d| \leq \kappa$ **then**
21:          $success =\leftarrow success + 1/n_{testrollout}$
---

## 5   Experiments

To evaluate our proposed method, we conducted experiments across three maze environments simulated in MuJoCo[3]: *PointUMaze*, *PointNMaze*, and *PointSpiralMaze*. In these environments, a goal is interpreted as the $(x, y)$ position of the agent achieved in an episode. These environments have been specifically chosen due to their structural characteristics, which are ideal for testing environment-agnostic curriculum generation strategies. Moreover, they present a variety of complex and diverse navigation challenges, requiring an agent to learn effective exploration and exploitation.

We compared our approach DICURL against nine state-of-the-art CRL algorithms, namely ACL [18], GOAL-GAN [14], HGG [19], ALP-GMM [17], VDS [16], PLR [15], CURROT [23], GRADI-ENT [13], and OUTPACE [12], each one run with 5 different random seeds. The primary conceptual differences between ours and baseline algorithms are summarized in Table 1. Details on the baseline algorithms, parametrization, training setup, and maze environments are given in Supp. Mat. C. Our codebase is available at: https://github.com/erdiphd/DiCuRL/.

The results, shown in Fig. 1[4] and detailed in Table 2[5], demonstrate the effectiveness of the proposed DICURL method. Notably, DICURL outperforms or matches all the baseline methods. For the PointNMaze and PointSpiralMaze, the success rate of all methods, except for ours and OUTPACE (and HGG for PointNMaze), is close to zero (detailed results are reported in Supp. Mat. C). ACL, GoalGAN, ALP-GMM, VDS, and PLR, which lack awareness of the target distribution, underperform compared to target-aware methods such as our proposed approach, OUTPACE, GRADIENT, CURROT, and HGG. HGG encounters difficulties because of infeasible curriculum proposals. This is a result of its reliance on the Euclidean distance metric.

Both CURROT and GRADIENT generate curriculum goals using the optimal transport method and the Wasserstein distance metric, relying on the geometry of the environment. This dependence might be the reason for their inconsistent and poor performance across the different environments.

OUTPACE, utilizes instead the Wasserstein distance and an uncertainty classifier for uncertainty-aware CRL, exhibiting similar performance to our approach but with slower convergence and higher variance in success rate. This is likely due to the fact that OUTPACE's curriculum heavily relies on the visited state distributions, which necessitate an initial exploration by the agent. While also our approach depends on visited states, incorporating the Qand AIM reward functions into curriculum generation facilitates exploration beyond them, potentially explaining the performance differences between DICURL and OUTPACE.

Lastly, we recall that our approach generates curriculum goals based on the reverse diffusion process, which allows the generated curriculum goals to gradually shift from the initial state distribution to the desired goal distribution. Fig. 2 shows an example of a curriculum set generated by DICURL, for the case of the PointUMaze environment, at each iteration of the reverse diffusion process. Fig. 3, instead, shows the differences between the curriculum goals generated by DICURL, GRADIENT, and HGG in the PointSpiralMaze environment: it can be seen how DICURL manages to generate curriculum goals that explore the whole environment more effectively than the baseline algorithms. Supp. Mat. D shows the curriculum goals generated in the other two environments and illustrates the dynamics of the diffusion process during training for the case of PointUMaze. To demonstrate the applicability of DICURL to robot manipulation tasks, we evaluated it on the FetchPush and FetchPickAndPlace tasks using a sparse reward setting. Further details can be found in Supp. Mat. E.

Table 1: Comparison of DICURL with previous CRL methods from the literature (sorted by year).

| Algorithm | Curriculum method | Target dist. curriculum | Geometry-agnostic | Off-policy | External reward | Venue, year |
|---|---|---|---|---|---|---|
| ACL [18] | LSTM | ✗ | ✓ | ✗ | ✗ | PMLR, 2017 |
| GoalGAN [14] | GAN | ✗ | ✓ | ✗ | ✗ | PMLR, 2018 |
| HGG [19] | $Q, \mathcal{B}, W_2$ | $\mathcal{G}^+$ | ✗ | ✓ | ✗ | NeurIPS, 2019 |
| ALP-GMM [17] | GMM | ✗ | ✓ | ✓ | ✗ | PMLR, 2020 |
| VDS [16] | $Q, \mathcal{B}$ | ✗ | ✓ | ✓ | ✗ | NeurIPS, 2020 |
| PLR [15] | TD-Error | ✗ | ✓ | ✗ | ✗ | PMLR, 2021 |
| CURROT [23] | $W_2$ | $\mathcal{G}^+, \mathcal{U}$ | ✗ | ✓ | ✗ | PMLR, 2022 |
| GRADIENT [13] | $W_2$ | $\mathcal{G}^+$ | ✗ | ✓ | ✓ | NeurIPS, 2022 |
| OUTPACE [12] | CNML | $\mathcal{G}^+$ | ✓ | ✓ | ✓ | ICLR, 2023 |
| DICURL (Ours) | Diffusion | $\mathcal{G}^+$ | ✓ | ✓ | ✓ | |

**Ablation Study** We conducted an ablation study to investigate the impact of the AIM reward function $r_\varphi$ and $Q_\phi$ function in generating curriculum goals with our method (DICURL). For that, we omitted, separately, the reward function $r_\varphi$ and the $Q_\phi$ function from Eq. 11, and plotted the success rate (with three different seeds) in Fig. 4a for the most challenging maze environment, PointSpiralMaze. The results indicate that the agent performs worse without the AIM reward function $r_\varphi$ and fails to achieve the task without the $Q_\phi$ function. The generated curriculum goals without the $r_\varphi$ or $Q_\phi$ function are shown in Fig. 4b and 4c, respectively. Fig. 4d, instead, illustrates the AIM reward value across different training episodes in a clockwise direction. Specifically, the first row and first column

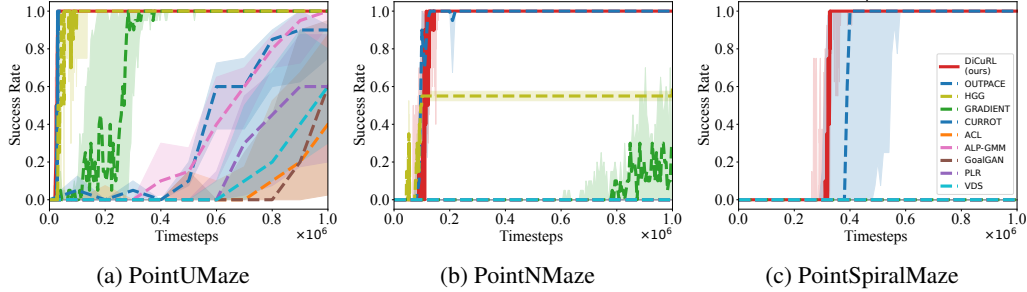

|(a) PointUMaze|(b) PointNMaze|(c) PointSpiralMaze|

Figure 1: Test success rate for the algorithms under comparison on the three maze tasks.

Table 2: No. of timesteps (rounded) to reach success rate $1.0$ (average $\pm$ std. dev. across 5 runs). NaN indicates that a success rate of $1.0$ is not reached within the maximum budget of 1e6 timesteps.

| Algorithm | PointUMaze | PointNMaze | PointSpiralMaze |
|---|---|---|---|
| DICURL (Ours) | $28333 \pm 3036$ | $108428 \pm 34718$ | $305833 \pm 43225$ |
| OUTPACE | $29800 \pm 4166$ | $113333 \pm 24267$ | $396875 \pm 111451$ |
| HGG | $48750 \pm 24314$ | NaN | NaN |
| GRADIENT | $263431 \pm 114795$ | NaN | NaN |

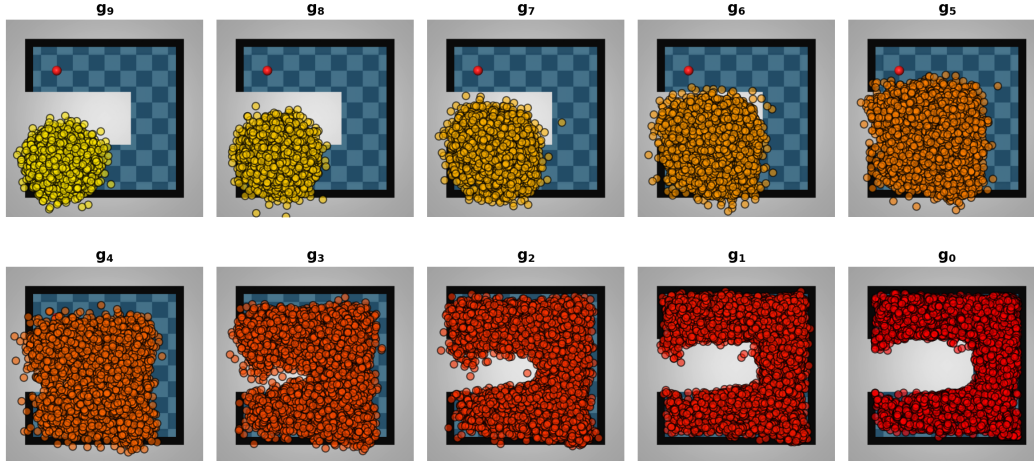

Figure 2: The curriculum goal set $\mathcal{G}_c$ generated by DICURL during the reverse diffusion process (lines 5-7 in Algorithm 1) for the PointUMaze environment. The color indicates the goals generated at a specific iteration step during the reverse diffusion process of the diffusion model. Then these goals are selected in Eq. (12) based on the given cost function.

in Fig. 4d represent the reward values at the very beginning of training. As training progresses, the reward values shift towards the left corner of the maze environment (1st row, 2nd column). In the middle of training, the reward values are concentrated around the left corner of the maze environment (2nd row, 2nd column), and, by the end of training, the reward values converge to the desired goal area (2nd row, 1st column). This progression explains why the generated curriculum goals are not guiding the agent effectively but are instead distributed in the corner points shown in Fig. 4c. We have also demonstrated the behavior of the AIM reward function across different training episodes in our Supp. Mat. D for the PointUMaze environment. Additionally, we examined the impact of SAC with a fixed initial state $[0, 0]$ and SAC with a random initial state. To do that, we removed the curriculum goal generation mechanism and assigned the desired goal, and then trained the agent using either SAC with a fixed initial state $[0, 0]$ or SAC with a random initial state. For the random initial state, we sampled goals randomly in the environment. To avoid starting the agent inside the maze walls, we performed an infeasibility check, resampling the initial state until it was feasible. We compared our approach using three different seeds with both the fixed initial state + SAC and the random initial state + SAC across all maze environments, and the success rates are shown in Fig. 5.

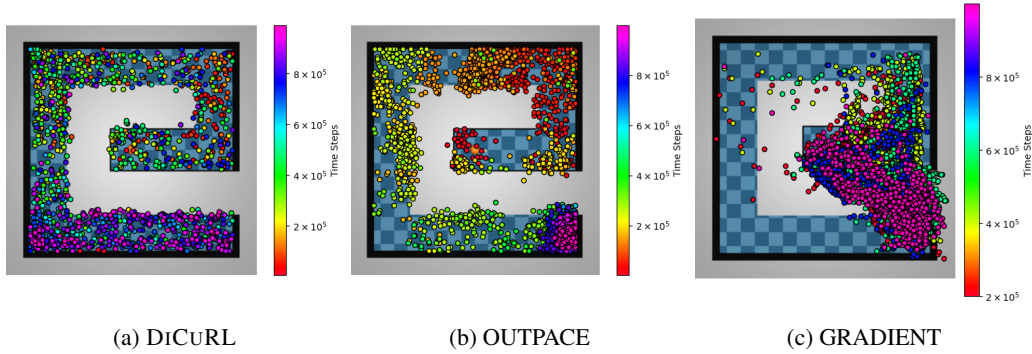

(a) DICURL &emsp; (b) OUTPACE &emsp; (c) GRADIENT

Figure 3: Curriculum goals generated by DICURL, GRADIENT, and HGG in the PointSpiralMaze environment. The colors ranging from red to purple indicate the curriculum goals across different episodes of the training, and the orange dot and red dot are the agent and desired goal, respectively.

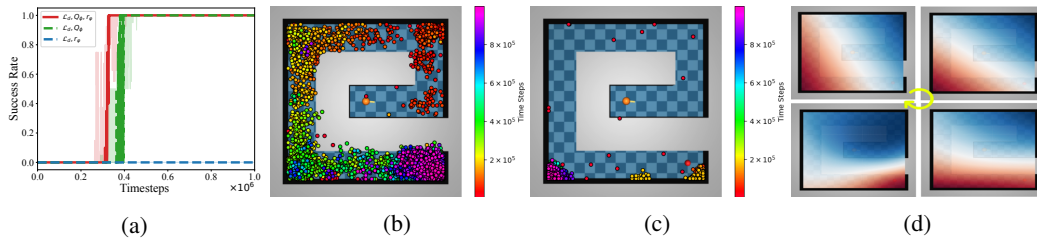

(a) &emsp; (b) &emsp; (c) &emsp; (d)

Figure 4: (a) Test success rate in the ablation study of DiCuRL on the PointSpiralMaze environment. (b) Curriculum goals when using only $Q_\phi$. (c) Curriculum goals when using only the AIM reward $r_\varphi$. (d) AIM reward $r_\varphi$ across different training episodes (around 60k, 100k, 250k and 600k timesteps), displayed clockwise from the top-left to the bottom-left quadrant.

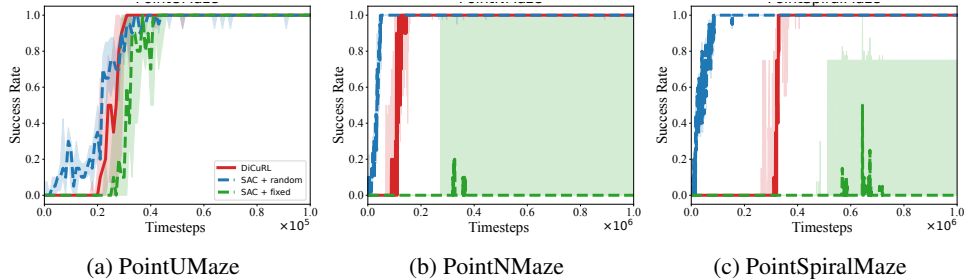

(a) PointUMaze &emsp; (b) PointNMaze &emsp; (c) PointSpiralMaze

Figure 5: Test success rate comparison between DiCuRL, SAC with random initial position, and SAC with fixed initial position. Note that the success rate for the PointUMaze environment in Fig. 5a is shown up to timestep $10^5$, whereas the others are shown up to $10^6$.

## 6 Conclusion and Limitations

In this work, we introduced DICURL, a novel approach that utilizes diffusion models to generate curriculum goals for an RL agent. The diffusion model is trained to minimize its loss function while simultaneously maximizing the expected value of $Q$ and the AIM reward function. The minimization of the diffusion loss helps capture the visited state distribution, while the maximization of $Q$ and AIM reward function helps generate goals at an appropriate difficulty level and at the same time guide the generated curriculum goals closer to the desired goal. Furthermore, the generated goals promote exploration due to the inherent noising and denoising mechanism of the diffusion model. Our proposed approach has two main limitations. First, while diffusion models excel at handling high-dimensional data, such as images [53], incorporating the AIM reward into a combined loss function can hinder the curriculum goal generation in such settings as the AIM reward may underperform in higher dimensionalities. Secondly, we employ the Minimum Cost Maximum Flow algorithm to select the optimal curriculum goals from the set generated by the diffusion model. However, alternative selection strategies could potentially be more effective. Future work will aim to address these limitations and extend our approach to more complex environments.

# 7 Acknowledgments

This study was supported by the CiLoCharging project, which is funded by the Bundesministerium für Wirtschaft und Klimaschutz (BMWK) of Germany under funding number 01ME20002C. Additionally, this work was supported by TUBITAK under the 2232 program, project number 121C148 ("LiRA"). We would like to express our gratitude to Arda Sarp Yenicesu for his assistance in writing this paper.

## Footnotes

[1]Details about deriving the conditional diffusion model equation are provided in Supp. Mat. A.

[2]In principle, instead of sampling experiences from the replay buffer and providing them to the diffusion model to generate a curriculum goal set, only the state of the last timestep, $s_T$, could be provided to the diffusion model to obtain a curriculum goal. The difference is that giving the sampled experiences to the diffusion model generates many curriculum points, while using only the last timestep $s_T$ generates only a single curriculum point, eliminating the need to apply any selection strategy. This is investigated in Supp. Mat. B.

[3]Details available at `https://robotics.farama.org/envs/maze/point_maze/`.

[4]A simplified version of the same figure is shown in Supp. Mat. F.

[5]We omit baselines unable to reach a success rate of 1.0 within the allotted timesteps on any of the environments.

[6]The reparameterization trick works by separating the deterministic and the stochastic parts of the sampling operation. Instead of directly sampling from the distribution $z \sim \mathcal{N}(\mu, \sigma)$, we can sample $\epsilon$ from the normal distribution $\mathcal{N}(0,1)$, multiply it by standard deviation $\sigma$, and add mean $\mu$: $z = \mu + \sigma\epsilon$

[7]As a reminder, in the proposed algorithm we sample a mini-batch $b$ (containing many states from different timesteps) from the replay buffer $\mathcal{B}$ and provide it to the curriculum goal generator (i.e., the diffusion model) to generate a curriculum goal distribution $\mathcal{G}_c$, and then select the optimal curriculum goal $g_c$ using bipartite graph optimization by maximizing the cost function given in Eq. 12.

# References

[1] Mnih, Volodymyr and Kavukcuoglu, Koray and Silver, David and Graves, Alex and Antonoglou, Ioannis and Wierstra, Daan and Riedmiller, Martin. Playing Atari with deep reinforcement learning, 2013. arXiv:1312.5602.

[2] Mnih, Volodymyr and Kavukcuoglu, Koray and Silver, David and Rusu, Andrei A and Veness, Joel and Bellemare, Marc G and Graves, Alex and Riedmiller, Martin and Fidjeland, Andreas K and Ostrovski, Georg and others. Human-level control through deep reinforcement learning. *Nature*, 518:529–533, 2015.

[3] Silver, David and Hubert, Thomas and Schrittwieser, Julian and Antonoglou, Ioannis and Lai, Matthew and Guez, Arthur and Lanctot, Marc and Sifre, Laurent and Kumaran, Dharshan and Graepel, Thore and others. A general reinforcement learning algorithm that masters chess, shogi, and Go through self-play. *Science*, 362(6419):1140–1144, 2018.

[4] Silver, David and Huang, Aja and Maddison, Chris J and Guez, Arthur and Sifre, Laurent and Van Den Driessche, George and Schrittwieser, Julian and Antonoglou, Ioannis and Panneershelvam, Veda and Lanctot, Marc and others. Mastering the game of Go with deep neural networks and tree search. *Nature*, 529:484–489, 2016.

[5] Rajeswaran, Aravind and Lowrey, Kendall and Todorov, Emanuel V and Kakade, Sham M. Towards generalization and simplicity in continuous control. In *Advances in Neural Information Processing Systems*, volume 30, San Diego, CA, USA, 2017. Neural Information Processing Systems Foundation.

[6] Ng, Andrew Y and Coates, Adam and Diel, Mark and Ganapathi, Varun and Schulte, Jamie and Tse, Ben and Berger, Eric and Liang, Eric. Autonomous inverted helicopter flight via reinforcement learning. In *International Symposium on Experimental Robotics*, pages 363–372, Berlin Heidelberg, Germany, 2006. Springer.

[7] Timothy P. Lillicrap and Jonathan J. Hunt and Alexander Pritzel and Nicolas Heess and Tom Erez and Yuval Tassa and David Silver and Daan Wierstra. Continuous control with deep reinforcement learning, 2019. arXiv:1509.02971.

[8] Sayar, Erdi and Bing, Zhenshan and D'Eramo, Carlo and Oguz, Ozgur S and Knoll, Alois. Contact Energy Based Hindsight Experience Prioritization, 2023. arXiv:2312.02677.

[9] Andrew Y Ng, Daishi Harada, and Stuart Russell. Policy invariance under reward transformations: Theory and application to reward shaping. In *International Conference on Machine Learning*, volume 99, pages 278–287. Citeseer, 1999.

[10] Rui Zhao and Volker Tresp. Energy-based hindsight experience prioritization. In Aude Billard, Anca Dragan, Jan Peters, and Jun Morimoto, editors, *Conference on Robot Learning*, volume 87 of *Proceedings of Machine Learning Research*, pages 113–122. PMLR, 29–31 Oct 2018.

[11] Rui Zhao, Xudong Sun, and Volker Tresp. Maximum entropy-regularized multi-goal reinforcement learning. In Kamalika Chaudhuri and Ruslan Salakhutdinov, editors, *International Conference on Machine Learning*, volume 97 of *Proceedings of Machine Learning Research*, pages 7553–7562. PMLR, 09–15 Jun 2019.

[12] Daesol Cho, Seungjae Lee, and H. Jin Kim. Outcome-directed reinforcement learning by uncertainty & temporal distance-aware curriculum goal generation. In *International Conference on Learning Representations*. OpenReview.net, 2023.

[13] Peide Huang, Mengdi Xu, Jiacheng Zhu, Laixi Shi, Fei Fang, and DING ZHAO. Curriculum reinforcement learning using optimal transport via gradual domain adaptation. In S. Koyejo, S. Mohamed, A. Agarwal, D. Belgrave, K. Cho, and A. Oh, editors, *Advances in Neural Information Processing Systems*, volume 35, pages 10656–10670. Curran Associates, Inc., 2022.

[14] Carlos Florensa, David Held, Xinyang Geng, and Pieter Abbeel. Automatic goal generation for reinforcement learning agents. In *International Conference on Machine Learning*, pages 1515–1528. PMLR, 2018.

[15] Minqi Jiang, Edward Grefenstette, and Tim Rocktäschel. Prioritized level replay. In Marina Meila and Tong Zhang, editors, *International Conference on Machine Learning*, volume 139 of *Proceedings of Machine Learning Research*, pages 4940–4950. PMLR, 18–24 Jul 2021.

[16] Yunzhi Zhang, Pieter Abbeel, and Lerrel Pinto. Automatic curriculum learning through value disagreement. In H. Larochelle, M. Ranzato, R. Hadsell, M.F. Balcan, and H. Lin, editors, *Advances in Neural Information Processing Systems*, volume 33, pages 7648–7659. Curran Associates, Inc., 2020.

[17] Rémy Portelas, Cédric Colas, Katja Hofmann, and Pierre-Yves Oudeyer. Teacher algorithms for curriculum learning of deep rl in continuously parameterized environments. In Leslie Pack Kaelbling, Danica Kragic, and Komei Sugiura, editors, *Conference on Robot Learning*, volume 100 of *Proceedings of Machine Learning Research*, pages 835–853. PMLR, 30 Oct–01 Nov 2020.

[18] Alex Graves, Marc G. Bellemare, Jacob Menick, Rémi Munos, and Koray Kavukcuoglu. Automated curriculum learning for neural networks. In Doina Precup and Yee Whye Teh, editors, *International Conference on Machine Learning*, volume 70 of *Proceedings of Machine Learning Research*, pages 1311–1320. PMLR, 06–11 Aug 2017.

[19] Ren, Zhizhou and Dong, Kefan and Zhou, Yuan and Liu, Qiang and Peng, Jian. Exploration via Hindsight Goal Generation. arXiv:1906.04279.

[20] Pascal Klink, Hany Abdulsamad, Boris Belousov, Carlo D'Eramo, Jan Peters, and Joni Pajarinen. A probabilistic interpretation of self-paced learning with applications to reinforcement learning. *Journal of Machine Learning Research*, 22(182):1–52, 2021.

[21] Pascal Klink, Carlo D' Eramo, Jan R Peters, and Joni Pajarinen. Self-paced deep reinforcement learning. In H. Larochelle, M. Ranzato, R. Hadsell, M.F. Balcan, and H. Lin, editors, *Advances in Neural Information Processing Systems*, volume 33, pages 9216–9227. Curran Associates, Inc., 2020.

[22] Pascal Klink, Hany Abdulsamad, Boris Belousov, and Jan Peters. Self-paced contextual reinforcement learning. In Leslie Pack Kaelbling, Danica Kragic, and Komei Sugiura, editors, *Conference on Robot Learning*, volume 100 of *Proceedings of Machine Learning Research*, pages 513–529. PMLR, 30 Oct–01 Nov 2020.

[23] Pascal Klink, Haoyi Yang, Carlo D'Eramo, Jan Peters, and Joni Pajarinen. Curriculum reinforcement learning via constrained optimal transport. In Kamalika Chaudhuri, Stefanie Jegelka, Le Song, Csaba Szepesvari, Gang Niu, and Sivan Sabato, editors, *International Conference on Machine Learning*, volume 162 of *Proceedings of Machine Learning Research*, pages 11341–11358. PMLR, 17–23 Jul 2022.

[24] Kevin Li, Abhishek Gupta, Ashwin Reddy, Vitchyr H Pong, Aurick Zhou, Justin Yu, and Sergey Levine. Mural: Meta-learning uncertainty-aware rewards for outcome-driven reinforcement learning. In Marina Meila and Tong Zhang, editors, *International Conference on Machine Learning*, volume 139 of *Proceedings of Machine Learning Research*, pages 6346–6356. PMLR, 18–24 Jul 2021.

[25] Meng Fang, Tianyi Zhou, Yali Du, Lei Han, and Zhengyou Zhang. Curriculum-guided hindsight experience replay. In H. Wallach, H. Larochelle, A. Beygelzimer, F. d'Alché-Buc, E. Fox, and R. Garnett, editors, *Advances in Neural Information Processing Systems*, volume 32. Curran Associates, Inc., 2019.

[26] Zhenshan Bing, Matthias Brucker, Fabrice O. Morin, Rui Li, Xiaojie Su, Kai Huang, and Alois Knoll. Complex robotic manipulation via graph-based hindsight goal generation. *IEEE Transactions on Neural Networks and Learning Systems*, 33(12):7863–7876, 2022.

[27] Zhenshan Bing, Hongkuan Zhou, Rui Li, Xiaojie Su, Fabrice O. Morin, Kai Huang, and Alois Knoll. Solving robotic manipulation with sparse reward reinforcement learning via graph-based diversity and proximity. *IEEE Transactions on Industrial Electronics*, 70(3):2759–2769, 2023.

[28] Michael Janner, Yilun Du, Joshua Tenenbaum, and Sergey Levine. Planning with diffusion for flexible behavior synthesis. In Kamalika Chaudhuri, Stefanie Jegelka, Le Song, Csaba Szepesvari, Gang Niu, and Sivan Sabato, editors, *International Conference on Machine Learning*, volume 162 of *Proceedings of Machine Learning Research*, pages 9902–9915. PMLR, 17–23 Jul 2022.

[29] Chang Chen, Fei Deng, Kenji Kawaguchi, Caglar Gulcehre, and Sungjin Ahn. Simple hierarchical planning with diffusion, 2024. arXiv:2401.02644.

[30] Zhendong Wang, Jonathan J Hunt, and Mingyuan Zhou. Diffusion policies as an expressive policy class for offline reinforcement learning, 2022. arXiv:2208.06193.

[31] Ishan Durugkar, Mauricio Tec, Scott Niekum, and Peter Stone. Adversarial intrinsic motivation for reinforcement learning. In M. Ranzato, A. Beygelzimer, Y. Dauphin, P.S. Liang, and J. Wortman Vaughan, editors, *Advances in Neural Information Processing Systems*, volume 34, pages 8622–8636. Curran Associates, Inc., 2021.

[32] Emanuel Todorov, Tom Erez, and Yuval Tassa. MuJoCo: A physics engine for model-based control. In *IEEE/RSJ International Conference on Intelligent Robots and Systems*, pages 5026–5033. IEEE, 2012.

[33] Sanmit Narvekar, Bei Peng, Matteo Leonetti, Jivko Sinapov, Matthew E Taylor, and Peter Stone. Curriculum learning for reinforcement learning domains: A framework and survey. *Journal of Machine Learning Research*, 21(181):1–50, 2020.

[34] Martin Riedmiller, Roland Hafner, Thomas Lampe, Michael Neunert, Jonas Degrave, Tom Wiele, Vlad Mnih, Nicolas Heess, and Jost Tobias Springenberg. Learning by playing solving sparse reward tasks from scratch. In *International Conference on Machine Learning*, pages 4344–4353. PMLR, 2018.

[35] Tim Hertweck, Martin Riedmiller, Michael Bloesch, Jost Tobias Springenberg, Noah Siegel, Markus Wulfmeier, Roland Hafner, and Nicolas Heess. Simple sensor intentions for exploration, 2020. arXiv:2005.07541.

[36] Ashvin V Nair, Vitchyr Pong, Murtaza Dalal, Shikhar Bahl, Steven Lin, and Sergey Levine. Visual reinforcement learning with imagined goals. *Advances in Neural Information Processing Systems*, 31, 2018.

[37] Dmytro Korenkevych, A Rupam Mahmood, Gautham Vasan, and James Bergstra. Autoregressive policies for continuous control deep reinforcement learning, 2019. arXiv:1903.11524.

[38] Marcin Andrychowicz, Filip Wolski, Alex Ray, Jonas Schneider, Rachel Fong, Peter Welinder, Bob McGrew, Josh Tobin, OpenAI Pieter Abbeel, and Wojciech Zaremba. Hindsight experience replay. In I. Guyon, U. Von Luxburg, S. Bengio, H. Wallach, R. Fergus, S. Vishwanathan, and R. Garnett, editors, *Advances in Neural Information Processing Systems*, volume 30. Curran Associates, Inc., 2017.

[39] Ian Goodfellow, Jean Pouget-Abadie, Mehdi Mirza, Bing Xu, David Warde-Farley, Sherjil Ozair, Aaron Courville, and Yoshua Bengio. Generative adversarial nets. In Z. Ghahramani, M. Welling, C. Cortes, N. Lawrence, and K.Q. Weinberger, editors, *Advances in Neural Information Processing Systems*, volume 27. Curran Associates, Inc., 2014.

[40] Tom Schaul, Daniel Horgan, Karol Gregor, and David Silver. Universal value function approximators. In Francis Bach and David Blei, editors, *International Conference on Machine Learning*, volume 37 of *Proceedings of Machine Learning Research*, pages 1312–1320, Lille, France, 07–09 Jul 2015. PMLR.

[41] Chelsea Finn, Pieter Abbeel, and Sergey Levine. Model-agnostic meta-learning for fast adaptation of deep networks. In Doina Precup and Yee Whye Teh, editors, *International Conference on Machine Learning*, volume 70 of *Proceedings of Machine Learning Research*, pages 1126–1135. PMLR, 06–11 Aug 2017.

[42] Yilun Du, Sherry Yang, Bo Dai, Hanjun Dai, Ofir Nachum, Josh Tenenbaum, Dale Schuurmans, and Pieter Abbeel. Learning universal policies via text-guided video generation. *Advances in Neural Information Processing Systems*, 36, 2024.

[43] Po-Chen Ko, Jiayuan Mao, Yilun Du, Shao-Hua Sun, and Joshua B Tenenbaum. Learning to act from actionless videos through dense correspondences, 2023. arXiv:2310.08576.

[44] Cheng Chi, Siyuan Feng, Yilun Du, Zhenjia Xu, Eric Cousineau, Benjamin Burchfiel, and Shuran Song. Diffusion policy: Visuomotor policy learning via action diffusion, 2023. arXiv:2303.04137.

[45] Moritz Reuss, Maximilian Li, Xiaogang Jia, and Rudolf Lioutikov. Goal-conditioned imitation learning using score-based diffusion policies, 2023. arXiv:2304.02532.

[46] Hsiang-Chun Wang, Shang-Fu Chen, Ming-Hao Hsu, Chun-Mao Lai, and Shao-Hua Sun. Diffusion model-augmented behavioral cloning, 2023. arXiv:2302.13335.

[47] Tim Pearce, Tabish Rashid, Anssi Kanervisto, Dave Bignell, Mingfei Sun, Raluca Georgescu, Sergio Valcarcel Macua, Shan Zheng Tan, Ida Momennejad, Katja Hofmann, et al. Imitating human behaviour with diffusion models, 2023. arXiv:2301.10677.

[48] Tuomas Haarnoja, Aurick Zhou, Pieter Abbeel, and Sergey Levine. Soft actor-critic: Off-policy maximum entropy deep reinforcement learning with a stochastic actor. In Jennifer Dy and Andreas Krause, editors, *International Conference on Machine Learning*, volume 80 of *Proceedings of Machine Learning Research*, pages 1861–1870. PMLR, 10–15 Jul 2018.

[49] Villani, Cédric and others. *Optimal transport: old and new*, volume 338. Springer, Berlin Heidelberg, Germany, 2009.

[50] Arjovsky, Martin and Chintala, Soumith and Bottou, Léon. Wasserstein generative adversarial networks. In *International Conference on Machine Learning*, pages 214–223, Sydney, Australia, 2017. PMLR.

[51] Jonathan Ho, Ajay Jain, and Pieter Abbeel. Denoising diffusion probabilistic models. In H. Larochelle, M. Ranzato, R. Hadsell, M.F. Balcan, and H. Lin, editors, *Advances in Neural Information Processing Systems*, volume 33, pages 6840–6851. Curran Associates, Inc., 2020.

[52] David M. Blei, Alp Kucukelbir, and Jon D. McAuliffe. Variational inference: A review for statisticians. *Journal of the American Statistical Association*, 112(518):859–877, April 2017.

[53] Robin Rombach, Andreas Blattmann, Dominik Lorenz, Patrick Esser, and Björn Ommer. High-resolution image synthesis with latent diffusion models. In *IEEE/CVF Conference on Computer Vision and Pattern Recognition*, pages 10684–10695, 2022.

[54] Jascha Sohl-Dickstein, Eric Weiss, Niru Maheswaranathan, and Surya Ganguli. Deep unsupervised learning using nonequilibrium thermodynamics. In Francis Bach and David Blei, editors, *International Conference on Machine Learning*, volume 37 of *Proceedings of Machine Learning Research*, pages 2256–2265, Lille, France, 07–09 Jul 2015. PMLR.

[55] Calvin Luo. Understanding diffusion models: A unified perspective, 2022. arXiv:2208.11970.

[56] John Schulman, Filip Wolski, Prafulla Dhariwal, Alec Radford, and Oleg Klimov. Proximal policy optimization algorithms, 2017. arXiv:1707.06347.

# Supplemental Material

## A    Conditional Diffusion Model

Diffusion models [51, 54] are usually defined as Markov chains and trained using variational inference. The forward diffusion process gradually adds Gaussian noise to the data, while the reverse diffusion process subtracts the learned noise from the data.

Given the goal data $\mathbf{g}_0$ sampled from the real goal distribution $\mathbf{g}_0 \sim \mathcal{G}$, we can define the forward diffusion process using Markov chains and add a small amount of Gaussian noise in each timestep $k$ according to a variance schedule $\{\beta_k \in (0,1)\}_{k=1}^N$, and generate a sequence of noisy goal data samples $\mathbf{g}_1, \ldots, \mathbf{g}_N$:

$$q\left(\mathbf{g}_{1:N} \mid \mathbf{g}_0\right) := \prod_{k=1}^N q\left(\mathbf{g}_k \mid \mathbf{g}_{k-1}\right) \tag{13}$$

$$q\left(\mathbf{g}_k \mid \mathbf{g}_{k-1}\right) := \mathcal{N}\left(\mathbf{g}_k; \sqrt{1-\beta_k}\mathbf{g}_{k-1}, \beta_k \boldsymbol{I}\right) .$$

As the index of the timestep $k$ increases, the goal sample $\mathbf{g}_0$ gradually loses its features and becomes an isotropic Gaussian noise when $N \to \infty$.

Instead of sampling data recursively for a given timestep $k$, we can sample $\mathbf{g}_k$ in a closed form using the reparameterization trick[6]:

$$\mathbf{g}_k \sim \mathcal{N}\left(\mathbf{g}_k; \sqrt{1-\beta_k}\mathbf{g}_{k-1}, \beta_k \boldsymbol{I}\right) . \tag{14}$$

Let $\alpha_k = 1 - \beta_k$ and $\bar{\alpha}_k = \prod_{i=1}^N \alpha_i$. We can write:

$$\mathbf{g}_k = \sqrt{\alpha_k}\mathbf{g}_{k-1} + \sqrt{1-\alpha_k}\epsilon_{k-1} \tag{15}$$

$$\mathbf{g}_k = \sqrt{\alpha_k \alpha_{k-1}}\mathbf{g}_{k-2} + \sqrt{1-\alpha_k}\epsilon_{k-1} + \sqrt{\alpha_k(1-\alpha_{k-1})}\epsilon_{k-2} . \tag{16}$$

We then merge two Gaussian distributions with different variance $\mathcal{N}(0,\sigma_1^2)$ and $\mathcal{N}(0,\sigma_2^2)$ into a new Gaussian distribution $\mathcal{N}(0,\sigma_1^2 + \sigma_2^2)$. Specifically, we can merge the two equations above, where $\sqrt{1-\alpha_k + \alpha_k(1-\alpha_{k-1})} = \sqrt{1-\alpha_k \alpha_{k-1}}$:

$$\mathbf{g}_k = \sqrt{\alpha_k \alpha_{k-1}}\mathbf{g}_{k-2} + \sqrt{1-\alpha_k \alpha_{k-1}}\bar{\epsilon}_{k-2} \tag{17}$$

$$\mathbf{g}_k = \sqrt{\prod_{i=1}^N \alpha_k}\mathbf{g}_0 + \sqrt{1 - \prod_{i=1}^N \alpha_k}\epsilon_0 \tag{18}$$

where $\epsilon_k \sim \mathcal{N}(0,1)$.

Now, we can write:

$$\mathbf{g}_k = \sqrt{\bar{\alpha}_k}\mathbf{g}_0 + \sqrt{1-\bar{\alpha}_k}\epsilon \tag{19}$$

$$q\left(\mathbf{g}_k \mid \mathbf{g}_0\right) = \mathcal{N}\left(\mathbf{g}_k; \sqrt{\bar{\alpha}_k}\mathbf{g}_0, (1-\bar{\alpha}_k)\boldsymbol{I}\right) \tag{20}$$

Using the equation above, we can sample data in a closed form at any arbitrary timestep $k$.

As mentioned in the main text, the reverse diffusion process aims to recover the original input data from the noisy (diffused) data. It learns to progressively reverse the diffusion process step by step, and approximates the joint distribution $p_\theta(\boldsymbol{x}_{0:T})$. This process is defined as:

$$p_\theta\left(\mathbf{g}_{0:N}\right) := p\left(\mathbf{g}_N\right) \prod_{k=1}^T p_\theta\left(\mathbf{g}_{k-1} \mid \mathbf{g}_k\right) \tag{21}$$

$$p_\theta\left(\mathbf{g}_{k-1} \mid \mathbf{g}_k\right) := \mathcal{N}\left(\mathbf{g}_{k-1}; \boldsymbol{\mu}_\theta\left(\mathbf{g}_k, k\right), \boldsymbol{\Sigma}_\theta\left(\mathbf{g}_k, k\right)\right)$$

The reverse conditional probability is tractable when conditioned by $g^0$.

By Bayes rule, we have:

$$q\left(\mathbf{g}_{k-1} \mid \mathbf{g}_k, \mathbf{g}_0\right) = \frac{q\left(\mathbf{g}_k \mid \mathbf{g}_{k-1}, \mathbf{g}_0\right) q\left(\mathbf{g}_{k-1} \mid \mathbf{g}_0\right)}{q\left(\mathbf{g}_k \mid \mathbf{g}_0\right)} . \tag{22}$$

As we assume that the diffusion model is a Markov chain, the future state depends only on the present state, namely:

$$q\left(\mathbf{g}_k \mid \mathbf{g}_{k-1}, \mathbf{g}_0\right) = q\left(\mathbf{g}_k \mid \mathbf{g}_{k-1}\right) = \mathcal{N}\left(\mathbf{g}_k; \sqrt{\alpha_k}\mathbf{g}_{k-1}, (1-\alpha_k)\boldsymbol{I}\right) \tag{23}$$

$$q\left(\mathbf{g}_{k-1} \mid \mathbf{g}_0\right) = \mathcal{N}\left(\mathbf{g}_{k-1}; \sqrt{\bar{\alpha}_{k-1}}\mathbf{g}_0, (1-\bar{\alpha}_{k-1})\boldsymbol{I}\right) \tag{24}$$

$$q\left(\mathbf{g}_k \mid \mathbf{g}_0\right) = \mathcal{N}\left(\mathbf{g}_k; \sqrt{\bar{\alpha}_k}\mathbf{g}_0, (1-\bar{\alpha}_k)\boldsymbol{I}\right) \tag{25}$$

$$q\left(\mathbf{g}_{k-1} \mid \mathbf{g}_k, \mathbf{g}_0\right) = \frac{\mathcal{N}\left(\mathbf{g}_k; \sqrt{\alpha_k}\mathbf{g}_{k-1}, (1-\alpha_k)\boldsymbol{I}\right)\mathcal{N}\left(\mathbf{g}_{k-1}; \sqrt{\bar{\alpha}_{k-1}}\mathbf{g}_0, (1-\bar{\alpha}_{k-1})\boldsymbol{I}\right)}{\mathcal{N}\left(\mathbf{g}_k; \sqrt{\bar{\alpha}_k}\mathbf{g}_0, (1-\bar{\alpha}_k)\boldsymbol{I}\right)} \tag{26}$$

where:

$$\mathcal{N}\left(\mathbf{g}_k; \sqrt{\alpha_k}\mathbf{g}_{k-1}, (1-\alpha_k)\boldsymbol{I}\right) = \frac{1}{(2\pi)^{n/2}|(1-\alpha_k)\boldsymbol{I}|^{1/2}} \exp\left(-\frac{1}{2}(\mathbf{g}_k - \sqrt{\alpha_k}\mathbf{g}_{k-1})^T \frac{\boldsymbol{I}}{(1-\alpha_k)}(\mathbf{g}_k - \sqrt{\alpha_k}\mathbf{g}_{k-1})\right) \tag{27}$$

$$\approx \exp\left(-\frac{(\mathbf{g}_k - \sqrt{\alpha_k}\mathbf{g}_{k-1})^2}{2(1-\alpha_k)\boldsymbol{I}}\right) \tag{28}$$

$$\mathcal{N}\left(\mathbf{g}_{k-1}; \sqrt{\bar{\alpha}_{k-1}}\mathbf{g}_0, (1-\bar{\alpha}_{k-1})\boldsymbol{I}\right) = \frac{1}{(2\pi)^{n/2}|(1-\bar{\alpha}_{k-1})\boldsymbol{I}|^{1/2}} \exp\left(-\frac{1}{2}(\mathbf{g}_{k-1} - \sqrt{\bar{\alpha}_{k-1}}\mathbf{g}_0)^T \frac{\boldsymbol{I}}{(1-\bar{\alpha}_{k-1})}(\mathbf{g}_{k-1} - \sqrt{\bar{\alpha}_{k-1}}\mathbf{g}_0)\right) \tag{29}$$

$$\approx \exp\left(-\frac{(\mathbf{g}_{k-1} - \sqrt{\bar{\alpha}_{k-1}}\mathbf{g}_0)^2}{2(1-\bar{\alpha}_{k-1})}\right) \tag{30}$$

$$\mathcal{N}\left(\mathbf{g}_k; \sqrt{\bar{\alpha}_k}\mathbf{g}_0, (1-\bar{\alpha}_k)\boldsymbol{I}\right) = \frac{1}{(2\pi)^{n/2}|(1-\bar{\alpha}_k)\boldsymbol{I}|^{1/2}} \exp\left(-\frac{1}{2}(\mathbf{g}_k - \sqrt{\bar{\alpha}_k}\mathbf{g}_0)^T \frac{\boldsymbol{I}}{(1-\bar{\alpha}_k)}(\mathbf{g}_k - \sqrt{\bar{\alpha}_k}\mathbf{g}_0)\right) \tag{31}$$

$$\approx \exp\left(-\frac{(\mathbf{g}_k - \sqrt{\bar{\alpha}_k}\mathbf{g}_0)^2}{2(1-\bar{\alpha}_k)}\right) \tag{32}$$

We then can write:

$$q\left(\mathbf{g}_{k-1} \mid \mathbf{g}_k, \mathbf{g}_0\right) \approx \exp\left(-\frac{1}{2\boldsymbol{I}}\left(\frac{(\mathbf{g}_k - \sqrt{\alpha_k}\mathbf{g}_{k-1})_2}{(1-\alpha_k)} + \frac{(\mathbf{g}_{k-1} - \sqrt{\bar{\alpha}_{k-1}}\mathbf{g}_0)_2}{1-\bar{\alpha}_{k-1}} - \frac{(\mathbf{g}_k - \sqrt{\bar{\alpha}_k}\mathbf{g}_0)_2}{1-\bar{\alpha}_k}\right)\right) \tag{33}$$

$$\approx \exp\left(-\frac{1}{2\boldsymbol{I}}\left(\frac{(\mathbf{g}_k - \sqrt{\alpha_k}\mathbf{g}_{k-1})^2}{(1-\alpha_k)} + \frac{(\mathbf{g}_{k-1} - \sqrt{\bar{\alpha}_{k-1}}\mathbf{g}_0)^2}{1-\bar{\alpha}_{k-1}} - \frac{(\mathbf{g}_k - \sqrt{\bar{\alpha}_k}\mathbf{g}_0)^2}{1-\bar{\alpha}_k}\right)\right) \tag{34}$$

$$\approx \exp\left(-\frac{1}{2\boldsymbol{I}}\left(\frac{(\mathbf{g}_k^2 - 2\sqrt{\alpha_k}\mathbf{g}_k\mathbf{g}_{k-1} + \alpha_k\mathbf{g}_{k-1}^2)}{(1-\alpha_k)} + \frac{(\mathbf{g}_{k-1}^2 - 2\sqrt{\bar{\alpha}_{k-1}}\mathbf{g}_{k-1}\mathbf{g}_0 + \bar{\alpha}_{k-1}\mathbf{g}_0^2)}{1-\bar{\alpha}_{k-1}} - \frac{(\mathbf{g}_k - \sqrt{\bar{\alpha}_k}\mathbf{g}_0)^2}{1-\bar{\alpha}_k}\right)\right) \tag{35}$$

$$= \exp\left(-\frac{1}{2\boldsymbol{I}}\left(\frac{(-2\sqrt{\alpha_k}\mathbf{g}_k\mathbf{g}_{k-1} + \alpha_k\mathbf{g}_{k-1}^2)}{(1-\alpha_k)} + \frac{(\mathbf{g}_{k-1}^2 - 2\sqrt{\bar{\alpha}_{k-1}}\mathbf{g}_{k-1}\mathbf{g}_0)}{1-\bar{\alpha}_{k-1}} + C\left(\mathbf{g}_k, \mathbf{g}_0\right)\right)\right) \tag{36}$$

$$\approx \exp\left(-\frac{1}{2\boldsymbol{I}}\left(-\frac{2\sqrt{\alpha_k}\mathbf{g}_k\mathbf{g}_{k-1}}{(1-\alpha_k)} + \frac{\alpha_k\mathbf{g}_{k-1}^2}{(1-\alpha_k)} + \frac{\mathbf{g}_{k-1}^2}{1-\bar{\alpha}_{k-1}} - \frac{2\sqrt{\bar{\alpha}_{k-1}}\mathbf{g}_{k-1}\mathbf{g}_0}{1-\bar{\alpha}_{k-1}}\right)\right) \tag{37}$$

$$= \exp\left(-\frac{1}{2\boldsymbol{I}}\left(\left(\frac{\alpha_k}{1-\alpha_k}+\frac{1}{1-\bar{\alpha}_{k-1}}\right)\mathbf{g}_{k-1}^2 - 2\left(\frac{\sqrt{\alpha_k}\mathbf{g}_k}{1-\alpha_k}+\frac{\sqrt{\bar{\alpha}_{k-1}}\mathbf{g}_0}{1-\bar{\alpha}_{k-1}}\right)\mathbf{g}_{k-1}\right)\right)$$

(38)

$$= \exp\left(-\frac{1}{2\boldsymbol{I}}\left(\left(\frac{\alpha_k(1-\bar{\alpha}_{k-1})+1-\alpha_k}{(1-\alpha_k)(1-\bar{\alpha}_{k-1})}+\right)\mathbf{g}_{k-1}^2 - 2\left(\frac{\sqrt{\alpha_k}\mathbf{g}_k}{1-\alpha_k}+\frac{\sqrt{\bar{\alpha}_{k-1}}\mathbf{g}_0}{1-\bar{\alpha}_{k-1}}\right)\mathbf{g}_{k-1}\right)\right)$$

(39)

$$= \exp\left(-\frac{1}{2\boldsymbol{I}}\left(\left(\frac{\alpha_k-\bar{\alpha}_k+1-\alpha_k}{(1-\alpha_k)(1-\bar{\alpha}_{k-1})}\right)\mathbf{g}_{k-1}^2 - 2\left(\frac{\sqrt{\alpha_k}\mathbf{g}_k}{1-\alpha_k}+\frac{\sqrt{\bar{\alpha}_{k-1}}\mathbf{g}_0}{1-\bar{\alpha}_{k-1}}\right)\mathbf{g}_{k-1}\right)\right)$$

(40)

$$= \exp\left(-\frac{1}{2\boldsymbol{I}}\left(\left(\frac{1-\bar{\alpha}_k}{(1-\alpha_k)(1-\bar{\alpha}_{k-1})}\right)\mathbf{g}_{k-1}^2 - 2\left(\frac{\sqrt{\alpha_k}\mathbf{g}_k}{1-\alpha_k}+\frac{\sqrt{\bar{\alpha}_{k-1}}\mathbf{g}_0}{1-\bar{\alpha}_{k-1}}\right)\mathbf{g}_{k-1}\right)\right)$$

(41)

$$= \exp\left(-\frac{1}{2\boldsymbol{I}}\left(\left(\frac{1-\bar{\alpha}_k}{(1-\alpha_k)(1-\bar{\alpha}_{k-1})}\right)\left(\mathbf{g}_{k-1}^2 - 2\left(\frac{\frac{\sqrt{\alpha_k}\mathbf{g}_k}{1-\alpha_k}+\frac{\sqrt{\bar{\alpha}_{k-1}}\mathbf{g}_0}{1-\bar{\alpha}_{k-1}}}{\frac{1-\bar{\alpha}_k}{(1-\alpha_k)(1-\bar{\alpha}_{k-1})}}\right)\mathbf{g}_{k-1}\right)\right)\right)$$

(42)

$$= \exp\left(-\frac{1}{2\boldsymbol{I}}\left(\left(\frac{1-\bar{\alpha}_k}{(1-\alpha_k)(1-\bar{\alpha}_{k-1})}\right)\left(\mathbf{g}_{k-1}^2 - 2\frac{\left(\frac{\sqrt{\alpha_k}\mathbf{g}_k}{1-\alpha_k}+\frac{\sqrt{\bar{\alpha}_{k-1}}\mathbf{g}_0}{1-\bar{\alpha}_{k-1}}\right)(1-\alpha_k)(1-\bar{\alpha}_{k-1})}{1-\bar{\alpha}_k}\mathbf{g}_{k-1}\right)\right)\right)$$

(43)

$$= \exp\left(-\frac{1}{2\boldsymbol{I}}\left(\left(\frac{1}{\frac{(1-\alpha_k)(1-\bar{\alpha}_{k-1})}{1-\bar{\alpha}_k}}\right)\left(\mathbf{g}_{k-1}^2 - 2\frac{\sqrt{\alpha_k}(1-\bar{\alpha}_{k-1})\mathbf{g}_k+\sqrt{\bar{\alpha}_{k-1}}(1-\alpha_k)\mathbf{g}_0}{1-\bar{\alpha}_k}\mathbf{g}_{k-1}\right)\right)\right)$$

(44)

$$\approx \mathcal{N}\left(\mathbf{g}_{k-1};\underbrace{\frac{\sqrt{\alpha_k}(1-\bar{\alpha}_{k-1})\mathbf{g}_k+\sqrt{\bar{\alpha}_{k-1}}(1-\alpha_k)\mathbf{g}_0}{1-\bar{\alpha}_k}}_{\mu(\mathbf{g}_k,\mathbf{g}_0)},\underbrace{\frac{(1-\alpha_k)(1-\bar{\alpha}_{k-1})}{1-\bar{\alpha}_k}}_{\Sigma(k)}\boldsymbol{I}\right)$$

(45)

where $C\left(\mathbf{g}_k,\mathbf{g}_0\right)$ is a constant term with respect to $\mathbf{g}_{k-1}$ and is calculated using only $\mathbf{g}_k$, $\mathbf{g}_0$, and $\alpha_k$.

The Eq. (45) shows that at every step $\mathbf{g}_{k-1}\sim q\left(\mathbf{g}_{k-1}\mid\mathbf{g}_k,\mathbf{g}_0\right)$ follows a normal distribution with mean $\mu(\mathbf{g}_k,\mathbf{g}_0)$ and variance $\Sigma(k)$.

Given this, the goal is to learn a denoising model $p_\theta\left(\mathbf{g}_{k-1}\mid\mathbf{g}_k\right)$ by approximating the ground-truth denoising transition $q\left(\mathbf{g}_{k-1}\mid\mathbf{g}_k,\mathbf{g}_0\right)$. This can be done by minimizing the KL divergence between $p_\theta\left(\mathbf{g}_{k-1}\mid\mathbf{g}_k\right)$ and $q\left(\mathbf{g}_{k-1}\mid\mathbf{g}_k,\mathbf{g}_0\right)$.

The KL divergence between two Gaussian distributions $\mathcal{N}(x;\mu_x,\Sigma_x)$ and $\mathcal{N}(y;\mu_y,\Sigma_y)$ is defined as:

$$D_{\mathrm{KL}}\left(\mathcal{N}(x;\mu_x,\Sigma_x)\mid\mid\mathcal{N}(y;\mu_y,\Sigma_y)\right)=\frac{1}{2}\left[\log\frac{|\Sigma_y|}{|\Sigma_x|}-d+\mathrm{tr}(\Sigma_y^{-1}\Sigma_x)+(\mu_y-\mu_x)^T\Sigma_y^{-1}(\mu_y-\mu_x)\right].$$

(46)

The KL divergence between the denoising model $p_\theta\left(\mathbf{g}_{k-1}\mid\mathbf{g}_k\right)$ and the ground-truth denoising transition $q\left(\mathbf{g}_{k-1}\mid\mathbf{g}_k,\mathbf{g}_0\right)$ is:

$$\arg\min_\theta D_{\mathrm{KL}}\left(q\left(\mathbf{g}_{k-1}\mid\mathbf{g}_k,\mathbf{g}_0\right)\mid\mid p_\theta\left(\mathbf{g}_{k-1}\mid\mathbf{g}_k\right)\right).$$

(47)

The learning of the denoising model can be written as:

$$\arg\min_\theta D_{\mathrm{KL}}\left(\mathcal{N}(\mathbf{g}_{k-1};\mu(\mathbf{g}_k,\mathbf{g}_0),\Sigma)\mid\mid\mathcal{N}(\mathbf{g}_{k-1};\mu_\theta,\Sigma)\right)$$

(48)

$$\arg\min_\theta\frac{1}{2}\left[\log\frac{|\Sigma|}{|\Sigma|}-d+\mathrm{tr}(\Sigma^{-1}\Sigma)+(\mu_\theta-\mu(\mathbf{g}_k,\mathbf{g}_0))^T\Sigma^{-1}(\mu_\theta-\mu(\mathbf{g}_k,\mathbf{g}_0))\right]$$

(49)

$$\arg\min_{\theta} \frac{1}{2} \left[ \log 1 - d + d + (\mu_\theta - \mu(\mathbf{g}_k, \mathbf{g}_0))^T \Sigma^{-1} (\mu_\theta - \mu(\mathbf{g}_k, \mathbf{g}_0)) \right] \tag{50}$$

$$\arg\min_{\theta} \frac{1}{2} \left[ (\mu_\theta - \mu(\mathbf{g}_k, \mathbf{g}_0))^T \Sigma^{-1} (\mu_\theta - \mu(\mathbf{g}_k, \mathbf{g}_0)) \right] \tag{51}$$

$$\arg\min_{\theta} \frac{1}{2} \left[ (\mu_\theta - \mu(\mathbf{g}_k, \mathbf{g}_0))^T (\boldsymbol{I})^{-1} (\mu_\theta - \mu(\mathbf{g}_k, \mathbf{g}_0)) \right] \tag{52}$$

$$\arg\min_{\theta} \frac{1}{2} \left[ \|\mu_\theta - \mu(\mathbf{g}_k, \mathbf{g}_0)\|_2^2 \right] . \tag{53}$$

We can rearrange Eq. (19) as follows:

$$\mathbf{g}_0 = \frac{\mathbf{g}_k - \sqrt{1 - \bar{\alpha}_k}\epsilon}{\sqrt{\bar{\alpha}_k}} \tag{54}$$

and rewrite $\mu(\mathbf{g}_k, \mathbf{g}_0)$ in Eq. (45) as follows:

$$\mu(\mathbf{g}_k, \mathbf{g}_0) = \frac{\sqrt{\alpha_k}(1 - \bar{\alpha}_{k-1})\mathbf{g}_k + \sqrt{\bar{\alpha}_{k-1}}(1 - \alpha_k)\frac{\mathbf{g}_k - \sqrt{1-\bar{\alpha}_k}\epsilon}{\sqrt{\bar{\alpha}_k}}}{1 - \bar{\alpha}_k} \tag{55}$$

where $\bar{\alpha}_k = \prod_{i=1}^N \alpha_i$.

Then:

$$\mu(\mathbf{g}_k, \mathbf{g}_0) = \frac{\sqrt{\alpha_k}(1 - \bar{\alpha}_{k-1})\mathbf{g}_k + \sqrt{\alpha_1\alpha_2\dots\alpha_{k-1}}(1 - \alpha_k)\frac{\mathbf{g}_k - \sqrt{1-\bar{\alpha}_k}\epsilon}{\sqrt{\alpha_1\alpha_2\dots\alpha_k}}}{1 - \bar{\alpha}_k} \tag{56}$$

$$= \frac{\sqrt{\alpha_k}(1 - \bar{\alpha}_{k-1})\mathbf{g}_k}{1 - \bar{\alpha}_k} + \frac{(1 - \alpha_k)\,\mathbf{g}_k}{(1 - \bar{\alpha}_k)\sqrt{\alpha_k}} - \frac{(1 - \alpha_k)\sqrt{1 - \bar{\alpha}_k}\epsilon}{(1 - \bar{\alpha}_k)\sqrt{\alpha_k}} \tag{57}$$

$$= \frac{\sqrt{\alpha_k}(1 - \bar{\alpha}_{k-1})\mathbf{g}_k}{1 - \bar{\alpha}_k} + \frac{(1 - \alpha_k)\,\mathbf{g}_k}{(1 - \bar{\alpha}_k)\sqrt{\alpha_k}} - \frac{(1 - \alpha_k)\sqrt{1 - \bar{\alpha}_k}\epsilon}{(1 - \bar{\alpha}_k)\sqrt{\alpha_k}} \tag{58}$$

$$= \left( \frac{\sqrt{\alpha_k}(1 - \bar{\alpha}_{k-1})}{1 - \bar{\alpha}_k} + \frac{(1 - \alpha_k)}{(1 - \bar{\alpha}_k)\sqrt{\alpha_k}} \right) \mathbf{g}_k - \frac{(1 - \alpha_k)\sqrt{1 - \bar{\alpha}_k}}{(1 - \bar{\alpha}_k)\sqrt{\alpha_k}}\epsilon \tag{59}$$

$$= \left( \frac{\alpha_k(1 - \bar{\alpha}_{k-1})}{(1 - \bar{\alpha}_k)\sqrt{\alpha_k}} + \frac{1 - \alpha_k}{(1 - \bar{\alpha}_k)\sqrt{\alpha_k}} \right) \mathbf{g}_k - \frac{(1 - \alpha_k)}{\sqrt{1 - \bar{\alpha}_k}\sqrt{\alpha_k}}\epsilon \tag{60}$$

$$= \left( \frac{\alpha_k - \bar{\alpha}_k + 1 - \alpha_k}{(1 - \bar{\alpha}_k)\sqrt{\alpha_k}} \right) \mathbf{g}_k - \frac{1 - \alpha_k}{\sqrt{1 - \bar{\alpha}_k}\sqrt{\alpha_k}}\epsilon \tag{61}$$

$$= \frac{1}{\sqrt{\alpha_k}}\mathbf{g}_k - \frac{1 - \alpha_k}{\sqrt{1 - \bar{\alpha}_k}\sqrt{\alpha_k}}\epsilon \tag{62}$$

We can model the denoising transition mean as follows:

$$\mu_\theta(\mathbf{g}_k, k) = \frac{1}{\sqrt{\alpha_k}}\mathbf{g}_k - \frac{1 - \alpha_k}{\sqrt{1 - \bar{\alpha}_k}\sqrt{\alpha_k}}\epsilon_\theta(\mathbf{g}_k, k) . \tag{63}$$

Given this, the minimization of the KL divergence can be written as:

$$\arg\min_{\theta} D_{\mathrm{KL}}\left(q\left(\mathbf{g}_{k-1} \mid \mathbf{g}_k, \mathbf{g}_0\right) \| p_\theta\left(\mathbf{g}_{k-1} \mid \mathbf{g}_k\right)\right) \tag{64}$$

$$\arg\min_{\theta} D_{\mathrm{KL}}\left(\mathcal{N}(\mathbf{g}_{k-1}; \mu(\mathbf{g}_k, \mathbf{g}_0), \Sigma) \| \mathcal{N}(\mathbf{g}_{k-1}; \mu_\theta, \Sigma)\right) \tag{65}$$

$$\arg\min_{\theta} \frac{1}{2} \left[ \left\| \frac{1}{\sqrt{\alpha_k}}\mathbf{g}_k - \frac{1 - \alpha_k}{\sqrt{1 - \bar{\alpha}_k}\sqrt{\alpha_k}}\epsilon_\theta(\mathbf{g}_k, k) - \frac{1}{\sqrt{\alpha_k}}\mathbf{g}_k + \frac{1 - \alpha_k}{\sqrt{1 - \bar{\alpha}_k}\sqrt{\alpha_k}}\epsilon \right\|_2^2 \right] \tag{66}$$

$$\arg\min_{\theta} \frac{1}{2} \left[ \left\| \frac{1 - \alpha_k}{\sqrt{1 - \bar{\alpha}_k}\sqrt{\alpha_k}}\epsilon - \frac{1 - \alpha_k}{\sqrt{1 - \bar{\alpha}_k}\sqrt{\alpha_k}}\epsilon_\theta(\mathbf{g}_k, k) \right\|_2^2 \right] \tag{67}$$

$$\arg\min_{\theta} \frac{1}{2} \frac{(1 - \alpha_k)^2}{(1 - \bar{\alpha}_k)\alpha_k} \left[ \|\epsilon - \epsilon_\theta(\mathbf{g}_k, k)\|_2^2 \right] . \tag{68}$$

Thus far, we have only modeled the goal distribution $p_\theta(\mathbf{g})$. To transform this into a conditional diffusion model, we can incorporate the state $s$ information at each diffusion timestep $k$ [55] in Eq. (21):

$$p_\theta\left(\mathbf{g}_{0:N}\right) := p\left(\mathbf{g}_N\right)\prod_{k=1}^{T} p_\theta\left(\mathbf{g}_{k-1} \mid \mathbf{g}_k s\right) \qquad (69)$$

$$p_\theta\left(\mathbf{g}_{k-1} \mid \mathbf{g}_k\right) := \mathcal{N}\left(\mathbf{g}_{k-1}; \boldsymbol{\mu}_\theta\left(\mathbf{g}_k, s, k\right), \boldsymbol{\Sigma}_\theta\left(\mathbf{g}_k, s, k\right)\right) \ .$$

To conclude, we can model the denoising transition mean by rewriting Eq. (70) based on the state $s$ as follows:

$$\mu_\theta\left(\mathbf{g}_k, s, k\right) = \frac{1}{\sqrt{\alpha_k}}\mathbf{g}_k - \frac{1-\alpha_k}{\sqrt{1-\bar{\alpha}_k}\sqrt{\alpha_k}}\epsilon_\theta(\mathbf{g}_k, s, k) \ . \qquad (70)$$

# B $\quad s_T$ Strategy

In contrast to selecting optimal curriculum goals via bipartite graph optimization, as shown in Eq. 12 in the main text[7] we performed an additional experiment where we considered only the state from the final timestep $T$, under the assumption that it is closer to achieving the desired goal. The state $s_T$ at the last timestep of each episode is then input into the diffusion model, which generates curriculum goals to be achieved in the subsequent episode. The resulting variant is detailed in Algorithm 3, with the RL training process described in Algorithm 4.

As demonstrated by this additional experiment, the assumption that the final state is closer to the desired goal does not always hold true. As a matter of fact, the state at the last timestep in some episodes might be even further away from the desired goal compared to the states in previous timesteps. Consequently, the curriculum goals generated by the diffusion model may not gradually shift from the initial position toward the desired goal. If the state at the last timestep is not progressively moving towards the desired goal, the curriculum goals generated by the diffusion model may jump to different areas within the maze environment. This can lead to a decrease in sample efficiency and a slower success rate. This issue is illustrated in Fig. 6, where it can be seen, that particularly in the case of PointSpiralMaze, the variant of our algorithm that makes use of this strategy (that we call "$s_T$ strategy") converges later than the main described in the main text.

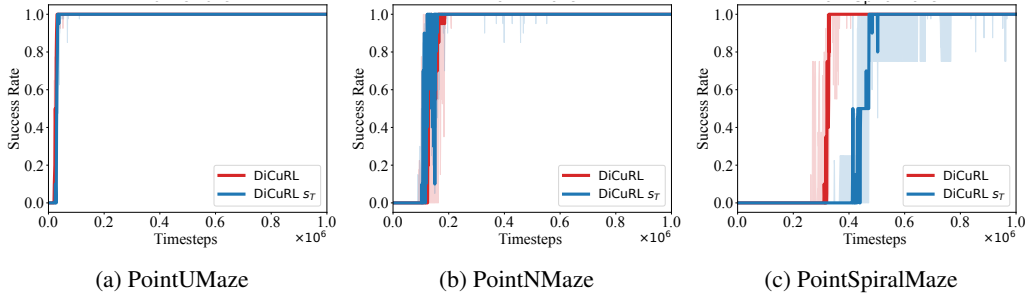

(a) PointUMaze $\qquad$ (b) PointNMaze $\qquad$ (c) PointSpiralMaze

Figure 6: Test success rate for the original DICURL algorithm described in the main text compared to a variant that makes use of the $s_T$ strategy.

**Algorithm 3** Diffusion Curriculum Goal Generator with the $s_T$ Strategy

---

1: **Input:** no. of reverse diffusion timestep $N$, training step $M$
2: **for** i $= 1, \ldots, M$ **do**                                                    ▷ Training iterations
3: $\quad$ $\mathbf{g}_N \sim \mathcal{N}(\mathbf{0}, \boldsymbol{I})$
4: $\quad$ **for** k $= N, \ldots, 1$ **do**                                            ▷ Reverse diffusion process
5: $\quad\quad$ $\epsilon \sim \mathcal{N}(\mathbf{0}, \boldsymbol{I})$
6: $\quad\quad$ $g_{k-1} = \frac{1}{\sqrt{\alpha_k}}\left(g_k - \frac{\beta_k}{\sqrt{1-\bar{\alpha}_k}}\epsilon_\theta(g_k, k, s_T)\right) + \sqrt{\beta_k}\epsilon$     ▷ using Eq. (9)
7: $\quad$ $k \sim \text{Uniform}(\{1, \ldots, N\})$
8: $\quad$ $\epsilon \sim \mathcal{N}(\mathbf{0}, \boldsymbol{I})$
9: $\quad$ Calculate diffusion $\mathcal{L}_d(\theta), Q(s, \mathbf{g}_0, \pi(s)), r(g_d, \mathbf{g}_0)$ ▷ Calculate with the generated goal $\mathbf{g}_0$
10: $\quad$ Calculate total loss $\mathcal{L} = \xi_d \mathcal{L}_d(\theta) - \xi_q Q(s, \mathbf{g}_0, \pi(s)) - \xi_r r(\mathbf{g}_0, g_d)$                ▷ Eq 11
11: $\quad$ $\theta \leftarrow \theta - \eta \nabla_\theta \mathcal{L}$                 ▷ Take gradient descent step and update the diffusion weights
$\quad$ **return** $\mathbf{g}_0$

---

---

**Algorithm 4** RL Training with the $s_T$ Strategy

---

1: **Input:** no. of episodes $E$, timesteps $T$
2: Select an off-policy algorithm $\mathbb{A}$                                          ▷ In our case, $\mathbb{A}$ is SAC
3: Initialize replay buffer $\mathcal{B} \leftarrow \emptyset$, $g_c \leftarrow g_d$ and networks $Q_\phi, \pi_\psi, r_\varphi$
4: **for** episode $= 0 \ldots E$ **do**
5: $\quad$ Sample initial state $s_0$
6: $\quad$ **for** $t = 0 \ldots T$ **do**
7: $\quad\quad$ $a_t = \pi(s_t, g_c)$
8: $\quad\quad$ Execute $a_t$, obtain next state $s_{t+1}$
9: $\quad\quad$ Store transition $(s_t, a_t, r_t, s_{t+1}, g_c)$ in $\mathcal{B}$
10: $\quad$ Sample a minibatch $b$ from replay buffer $\mathcal{B}$
11: $\quad$ $g_c \leftarrow \texttt{DiffusionCurriculumGenerator}(s_T)$
12: $\quad$ Update $Q$ and $\pi$ with $b$ to minimize $\mathcal{L}_Q$ and $\mathcal{L}_\pi$ in Eq. (1) and in Eq. (2)
13: $\quad$ Update the AIM reward function $r_\varphi$
14: $success \leftarrow 0$                                                              ▷ Success Rate
15: Sample a desired goal $g_d \sim \mathcal{G}$
16: **for** i $= 1 \ldots n_{testrollout}$ **do**
17: $\quad$ $a_t = \pi(s_t, g_d)$
18: $\quad$ Execute $a_t$, obtain next state $s_{t+1}$ and reward $r_t$
19: $\quad$ **if** $|\phi(s_{t+1}) - g_d| \leq \kappa$ **then**
20: $\quad\quad$ $success =\leftarrow success + 1/n_{testrollout}$

---

## C  Experimental Details

For implementing DICURL, we utilized the original implementation of OUTPACE [12] and augmented this codebase with our diffusion model for curriculum goal generation. We conducted our experiments on a cluster computer using an NVIDIA RTX A5000 GPU, 64GB of RAM, and a 4-core CPU. For the PointSpiralMaze environment, the total compute time of the main method and the $s_T$ strategy is approximately 23 hours and 5 hours, respectively. For the remaining environments, the main method and the $s_T$ strategy require approximately 11.5 hours and 2.5 hours, respectively.

**Baselines.**  The baseline CRL algorithms are trained as follows:

- HGG [19]: We utilized the default settings from the original implementation, which can be found at `https://github.com/Stilwell-Git/Hindsight-Goal-Generation`.

- CURROT [23]: We adhered to the default settings of the original implementation, available at `https://github.com/psclklnk/currot`.

- ALP-GMM [17], VDS [16], PLR [15], ACL [18], GoalGAN [14]: We followed the default settings from the implementation available at `https://github.com/psclklnk/currot`.

- GRADIENT [13]: We used the default settings from the implementation available at `https://github.com/PeideHuang/gradient`.

- OUTPACE [12]: We implemented the default settings from the implementation available at `https://github.com/jayLEE0301/outpace_official`.

**Training details.** All baseline models, except for the GRADIENT method, were trained using the Soft Actor-Critic (SAC) [48]. Although the original implementations of these algorithms primarily use on-policy methods, adaptations were made in the CURROT repository to utilize the off-policy SAC algorithm. This modification, available in the CURROT repository, allows for a more direct comparison of sample efficiency across all models. The GRADIENT method was trained using both the SAC and Proximal Policy Optimization (PPO) algorithms [56]. However, we present only the results obtained with the PPO algorithm, as it outperforms SAC in the three considered maze environments. All our algorithms' hyperparameters used in the experiments are reported in Table 3.

Table 3: Hyperparameters for DICURL.

| Parameter | Value | Parameter | Value |
|---|---|---|---|
| Critic hidden dim | 512 | Discount factor $\gamma_r$ | 0.99 |
| Critic hidden depth | 3 | $r_\varphi$ Update frequency | 1000 |
| Critic target $\tau$ | 0.01 | No. of gradient steps for $r_\varphi$ update | 10 |
| Critic target update frequency | 2 | No. of ensemble networks for $r_\varphi$ | 5 |
| Actor hidden dim | 512 | learning rate for $r_\varphi$ | 1e-4 |
| Actor hidden depth | 3 | RL optimizer | Adam |
| Actor update frequency | 2 | Diffusion network updates per episode | 25 |
| RL batch size | 512 | Diffusion loss coefficient $\xi_d$ | 1 |
| Init. temperature $\alpha_{\text{init}}$ of SAC | 0.3 | $Q$-function coefficient $\xi_q$ | 10 |
| Replay buffer $\mathcal{B}$ size | 1e6 | AIM reward function coefficient $\xi_r$ | 1 |
| Diffusion training iterations | 300 | Reverse diffusion timesteps | 10 |
| Diffusion learning rate | 3e-4 | Diffusion loss type | $l_2$ |
| Diffusion update frequency | 2500 | No. of training timesteps | 1e6 |

**Environment details.** In each task, the agent's state $s$ is represented as a vector, comprising its position, velocity along the $x$ and $y$ axes, orientation angle, and angular velocity around the $z$-axis. The agent's actions are determined by its velocity and angular velocity. The agent starts each episode from an initial state of $[0, 0]$. The desired goal distribution is derived by introducing uniform noise to the desired goal position. The desired goal positions are set to $[0, 8]$, $[8, 16]$, and $[8, -8]$, while the dimensions of the map are $12 \times 12$, $12 \times 20$, $20 \times 20$, respectively for the PointUMaze, the PointNMaze, and the PointSpiralMaze task. A task is considered successful when the agent reaches the sampled desired goal within a threshold distance of $0.5$.

**Detailed results.** We report in Table 4 the detailed results in terms of no. of timesteps needed to reach a success rate of $0.3$ for all the algorithms under comparison. It can be seen that most of the algorithms do not reach this rate within the given budget.

# D  Generated Curriculum Goals

Fig. 7 and Fig. 8 show the differences between the curriculum goals generated by DICURL, OUTPACE, GRADIENT, and HGG, respectively in the PointUMaze and PointNMaze environment. As shown in the case of PointSpiralMaze reported in the main text, also in these two environments DICURL manages to generate curriculum goals that explore the whole environment more effectively than GRADIENT and HGG.

## D.1  Dynamics of the Diffusion Process during Training

Considering the case of the PointUMaze environment, the curriculum goal sets, denoted as $\mathbf{g}_9, \mathbf{g}_8, \ldots, \mathbf{g}_0$ in Fig. 10, generated during the initial phase of training ($\sim 2000$ timesteps), are

Table 4: No. of timesteps (rounded) to reach success rate 0.3 (average $\pm$ std. dev. across 5 runs). NaN indicates that a success rate of 0.3 is not reached within the maximum budget of 1e6 timesteps.

| Algorithm | PointUMaze | PointNMaze | PointSpiralMaze |
|---|---|---|---|
| DICURL (Ours) | $25000 \pm 3958$ | $99000 \pm 32031$ | $305 \pm 43225$ |
| OUTPACE | $27600 \pm 4176$ | $106666 \pm 25603$ | $396875 \pm 111451$ |
| HGG | $34750 \pm 14956$ | $68000 \pm 15000.0$ | NaN |
| GRADIENT | $166986 \pm 86132$ | $755573 \pm 65967$ | NaN |
| CURROT | $700000 \pm 234520$ | NaN | NaN |
| ALP-GMM | $660000 \pm 205912$ | NaN | NaN |
| GoalGAN | $960000 \pm 290516$ | NaN | NaN |
| PLR | $833333 \pm 253859$ | NaN | NaN |
| VDS | $838461 \pm 294927$ | NaN | NaN |
| ACL | $1000000 \pm 316227$ | NaN | NaN |

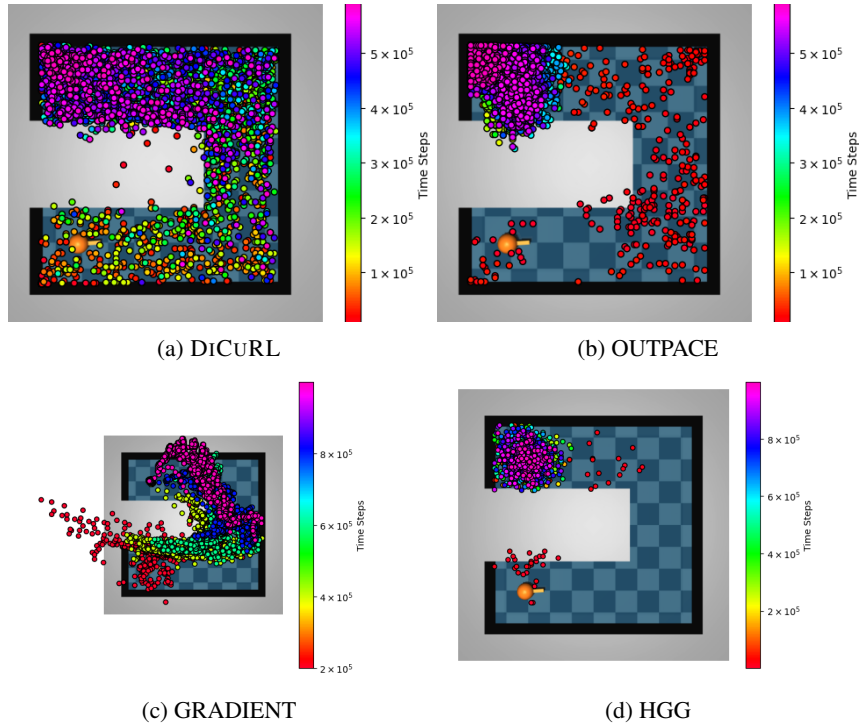

(a) DICURL

(b) OUTPACE

(c) GRADIENT

(d) HGG

Figure 7: Curriculum goals generated by DICURL, OUTPACE, GRADIENT, and HGG in the PointUMaze environment. The colors ranging from red to purple indicate the curriculum goals across different episodes of the training, and the orange dot and red dot are the agent and desired goal, respectively.

obtained through the reverse diffusion process. This process starts from Gaussian noise $\mathbf{g}_9$ and culminates in the final curriculum goal set $\mathbf{g}_0$, as expressed in lines 5 and 7 in Algorithm 1. It can be observed that the final curriculum goal set $\mathbf{g}_0$ is generated diagonally. This pattern arises due to the combined loss function of the AIM reward and the $Q$-value, along with the diffusion loss, as expressed in Eq. (11).

The AIM reward function, relative to the desired goal $g_d$, and the $Q$-value, derived from a sampled mini-batch $b$ from the replay buffer $\mathcal{B}$ at timestep $\sim 2000$, are depicted in Fig. 11a and Fig. 11b, respectively, using a 10-level contour plot. It can be observed that the reward values increase diagonally, while the $Q$-value has not yet shown a discernible pattern. As a result, the curriculum goal set generated by the diffusion model exhibits a pattern similar to the reward value.

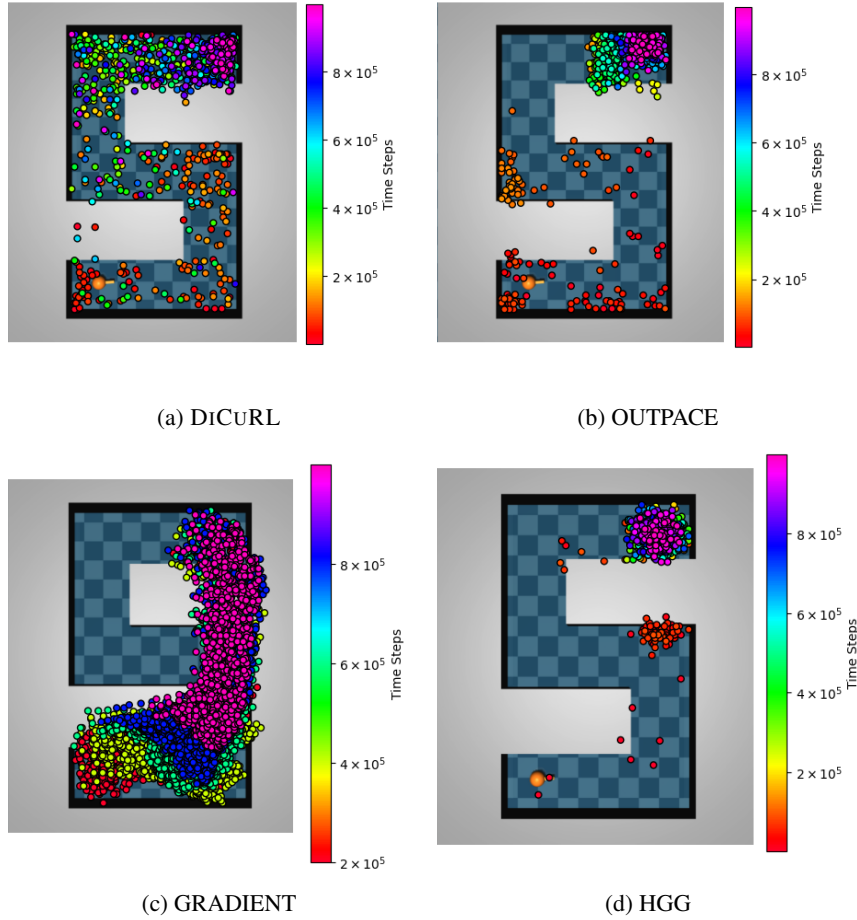

<center>(a) DICURL         (b) OUTPACE</center>

<center>(c) GRADIENT         (d) HGG</center>

Figure 8: Curriculum goals generated by DICURL, OUTPACE, GRADIENT, and HGG in the PointNMaze environment. The colors ranging from red to purple indicate the curriculum goals across different episodes of the training, and the orange dot and red dot are the agent and desired goal, respectively.

In the mid-training stage ($\sim$15000 timesteps), the final curriculum goal set $\mathbf{g}_0$ reflects the explored region of the environment, extending up to the top right corner, as shown in Fig. 12. This behavior aligns with the agent's exploration progress.

The AIM reward function in Fig. 13a exhibits higher and better-converged values compared to Fig. 11a, particularly in the top right corner. Similarly, the $Q$-value in Fig. 13b demonstrates its highest values concentrated from the middle right side to the top right corner. These observations suggest that the final curriculum goal set $\mathbf{g}_0$ has indeed been formed, encompassing the explored area up to the top right corner.

In the near-optimal policy learning phase ($\sim$ 30000 timesteps), the final curriculum goal set $\mathbf{g}_0$ encompasses the entire environment, extending from the initial position to the desired goal, as depicted in Fig. 14. This behavior aligns with the agent's progress towards achieving the optimal policy.

The AIM reward function in Fig. 15a appears to be converging to the desired goal more slowly. The $Q$-value in Fig. 15b demonstrates its highest values concentrated around the desired goal area. These observations suggest that the final curriculum goal set $\mathbf{g}_0$ has indeed been formed, encompassing the entire environment. Furthermore, it appears that the $Q$-value converges to the desired goal more rapidly than the AIM reward function. This observation aligns with our choice of a higher hyperparameter coefficient $\xi_q$ for the $Q$-value function compared to hyperparameter coefficient $\xi_r$

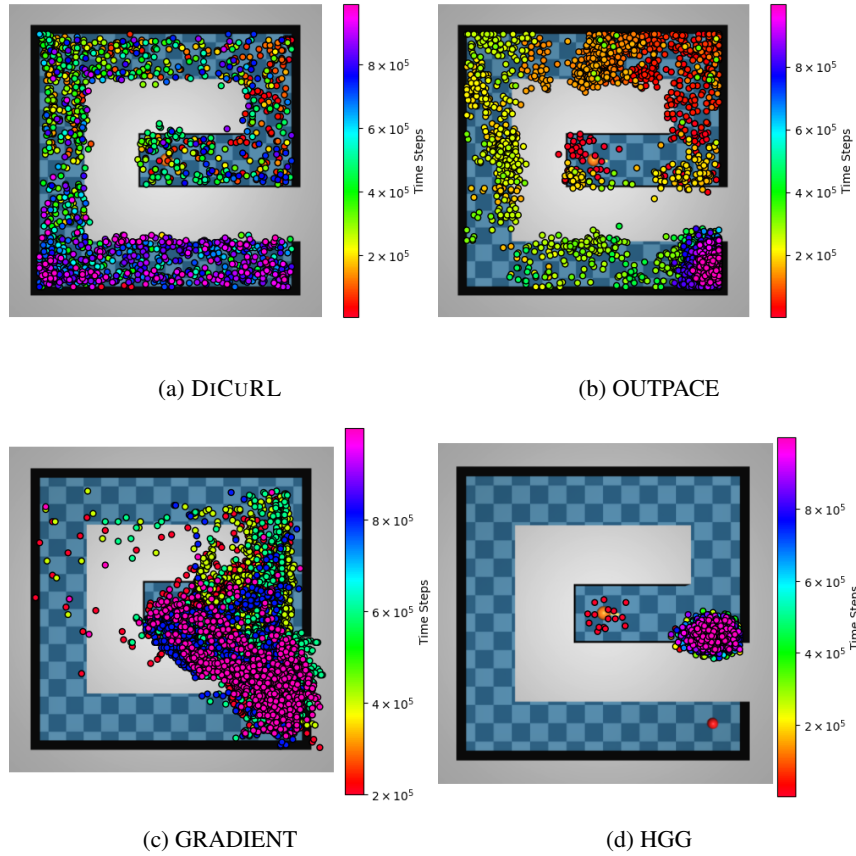

Figure 9: Curriculum goals generated by DICURL, GRADIENT, and HGG in the PointSpiralMaze environment. The colors ranging from red to purple indicate the curriculum goals across different episodes of the training, and the orange dot and red dot are the agent and desired goal, respectively.

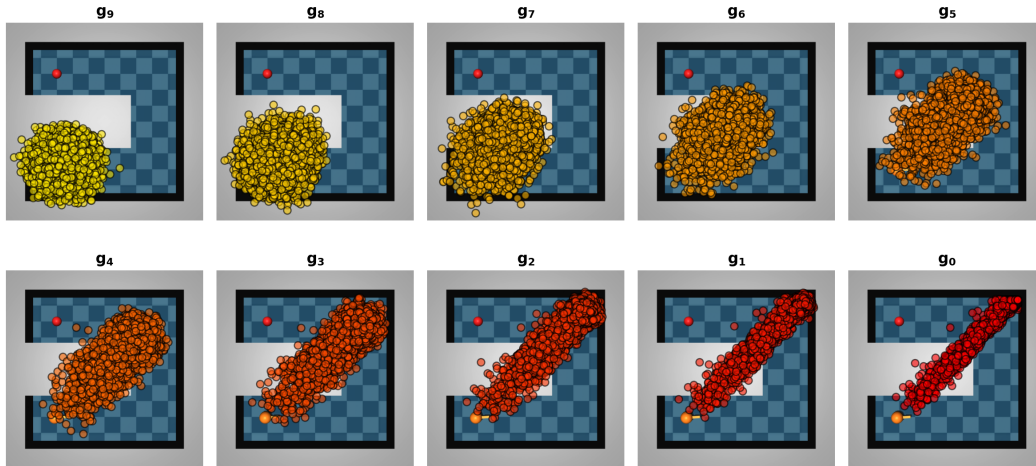

Figure 10: The curriculum goal set $\mathcal{G}_c$ generated by DICURL during the reverse diffusion process in the early stage of the training ($\sim$2000 timesteps) for the PointUMaze environment.

for the reward function in the total loss function shown in Eq. (11). This prioritizes the $Q$-value's influence on shaping the curriculum goals, promoting faster convergence towards the desired behavior.

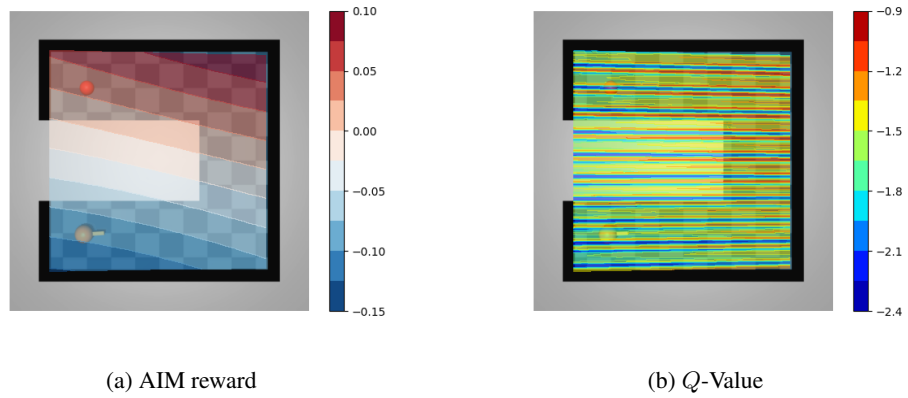

(a) AIM reward            (b) $Q$-Value

Figure 11: Visualization of the AIM reward function and $Q$-function in the early stage of the training ($\sim$2000 timesteps) for the PointUMaze environment.

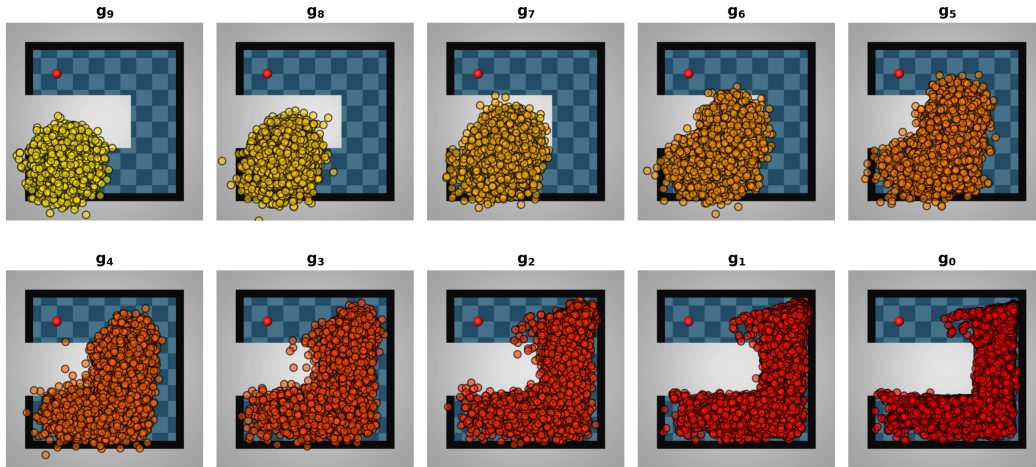

Figure 12: The curriculum goal set $\mathcal{G}_c$ generated by DICURL during the reverse diffusion process in the middle stage of the training ($\sim$15000 timesteps) for the PointUMaze environment.

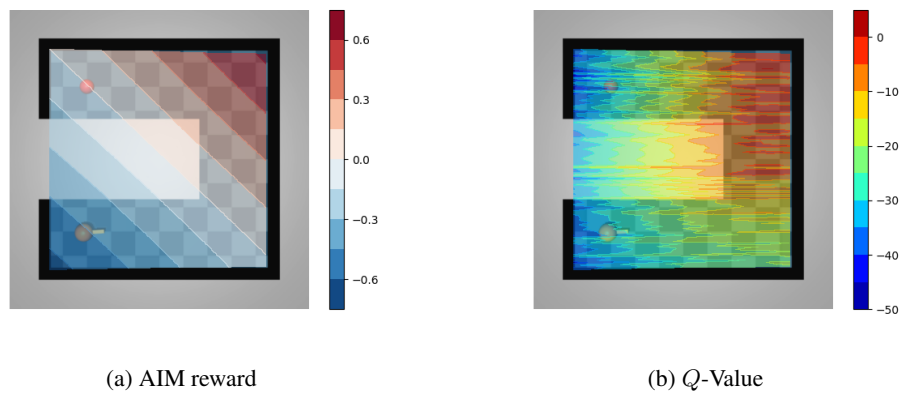

(a) AIM reward            (b) $Q$-Value

Figure 13: Visualization of the AIM reward function and $Q$-function in the middle stage of the training ($\sim$15000 timesteps) for the PointUMaze environment.

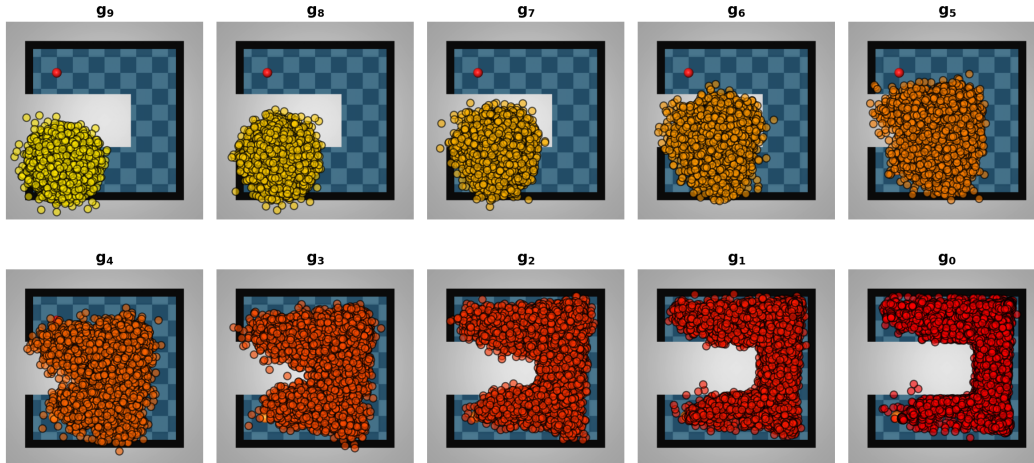

Figure 14: The curriculum goal set $\mathcal{G}_c$ generated by DICURL during the reverse diffusion process in the near-optimal stage of the training (∼30000 timesteps) for the PointUMaze environment.

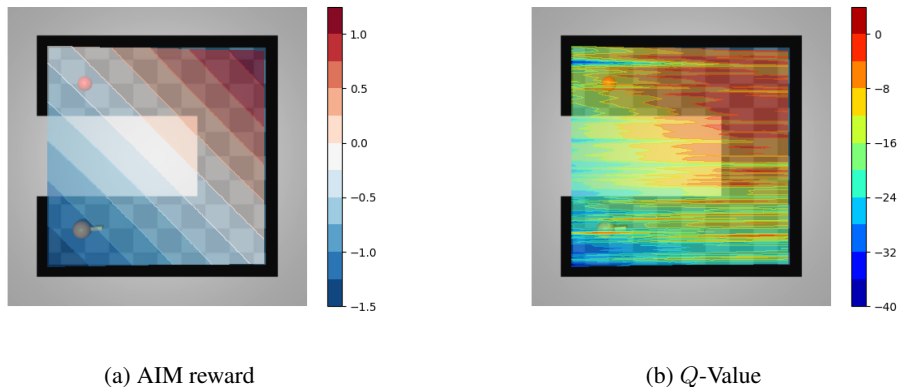

(a) AIM reward

(b) $Q$-Value

Figure 15: Visualization of the AIM reward function and $Q$-function in the near-optimal stage of the training (∼30000 timesteps) for the PointUMaze environment.

## E  Robot Manipulation Tasks

To demonstrate the applicability of our method to different tasks, particularly in robot manipulation tasks, we implemented our approach using the original HGG algorithm [19] repository, which can be found at `https://github.com/Stilwell-Git/Hindsight-Goal-Generation`. We converted the HGG code from TensorFlow to PyTorch to integrate it with our diffusion model, which is based on PyTorch. We selected two robot manipulation tasks, FetchPush and FetchPickAndPlace, and increased the environment difficulty by expanding the desired goal area. This is shown in Fig. 16c and 16d, where the yellow area indicates the object sampling region and the blue area indicates the desired goal sampling region. The action space is four-dimensional, where three dimensions represent the Cartesian displacement of the end effector, and the last dimension controls the opening and closing of the gripper. The state space is 25-dimensional, including the end-effector position, position, and rotation of the object, the linear and angular velocity of the object, and the left and right gripper velocity. More detailed information regarding the action space and observation space of these robotic tasks can be found in the Gymnasium library documentation. For these additional experiments, we compared our method with HGG and HER using the DDPG algorithm, to ensure alignment with the baselines, using five different seeds. The results are shown in Fig. 16a and Fig. 16b, respectively for FetchPush and FetchPickAndPlace. Note that in this setting, all RL algorithms (including ours) use a binary reward (i.e., a sparse reward). However, since our curriculum goal

generation algorithm is based on the AIM reward, we implemented the AIM reward function solely to generate curriculum goals while still using the sparse reward setting to train the DDPG algorithm using the diffusion model to generate curriculum goals.

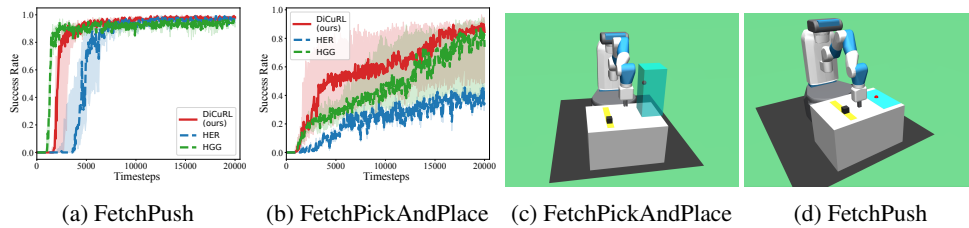

(a) FetchPush     (b) FetchPickAndPlace     (c) FetchPickAndPlace     (d) FetchPush

Figure 16: (a, b) Test success rate with solid lines indicating mean success rates and shaded areas variability over five seeds. (c, d) Overview of the two robot manipulation tasks. Blue and yellow regions denote the sampling areas for objects and goals, respectively.

Code available at: https://github.com/erdiphd/DiCuRL/tree/robot_manipulation

# F    Test Success Rate in Maze Tasks (Simplified Visualization)

In Fig. 1 in the main text, we presented the success rate of ten different algorithms (including ours) across the three maze tasks. To facilitate the visual comparison of the algorithms' performance, we provide in Fig. 17 a simplified visualization that reduces the line overlap by showing only the four top-performing algorithms (including ours).

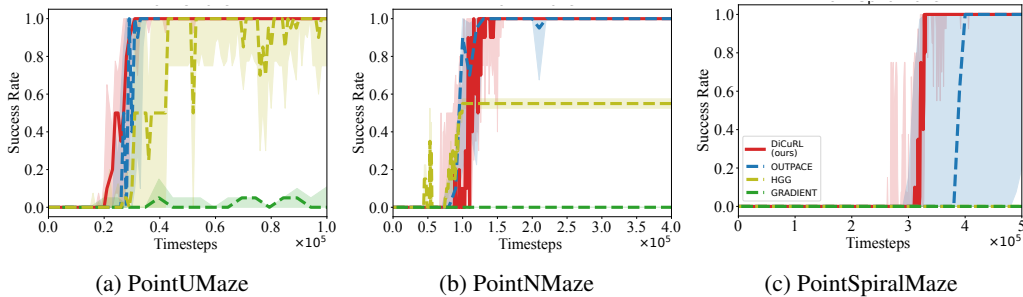

(a) PointUMaze     (b) PointNMaze     (c) PointSpiralMaze

Figure 17: Test success rate for the algorithms under comparison on the three maze tasks.

